# Globally Q-linear Gauss-Newton Method for Overparameterized Non-convex Matrix Sensing

**Xixi Jia**[1][*]   **Fangchen Feng**[2]   **Deyu Meng**[3,4]   **Defeng Sun**[5][†]

[1]School of Mathematics and Statistics, Xidian University
[2]L2TI Laboratory, University Sorbonne Paris Nord
[3] School of Mathematics and Statistics, Xi'an Jiaotong University
[4] Macao Institute of Systems Engineering, Macau University of Science and Technology
[5] Department of Applied Mathematics, The Hong Kong Polytechnic University
`hsijiaxidian@gmail.com`, `fangchen.feng@univ-paris13.fr`,
`dymeng@xjtu.edu.cn`, `defeng.sun@polyu.edu.hk`

## Abstract

This paper focuses on the optimization of overparameterized, non-convex low-rank matrix sensing (LRMS)—an essential component in contemporary statistics and machine learning. Recent years have witnessed significant breakthroughs in first-order methods, such as gradient descent, for tackling this non-convex optimization problem. However, the presence of numerous saddle points often prolongs the time required for gradient descent to overcome these obstacles. Moreover, overparameterization can markedly decelerate gradient descent methods, transitioning its convergence rate from linear to sub-linear. In this paper, we introduce an approximated Gauss-Newton (AGN) method for tackling the non-convex LRMS problem. Notably, AGN incurs a computational cost comparable to gradient descent per iteration but converges much faster without being slowed down by saddle points. We prove that, despite the non-convexity of the objective function, AGN achieves Q-linear convergence from random initialization to the global optimal solution. The global Q-linear convergence of AGN represents a substantial enhancement over the convergence of the existing methods for the overparameterized non-convex LRMS. The code for this paper is available at `https://github.com/hsijiaxidian/AGN`.

## 1   Introduction

Matrix sensing aims to recover an unknown low-rank matrix $M \in \mathbb{R}^{n \times n}$ from its linear measurement $\boldsymbol{b} = \mathcal{A}(M)$. Here each elements $\boldsymbol{b}_i$ is defined as $\boldsymbol{b}_i = \langle A_i, M \rangle$, with $i = 1, \cdots, m$, and $\mathcal{A}(\cdot)$ is a nearly isometric linear operator. It holds significance not only in practical applications but also in the realm of non-convex optimization [1–6]. As the problem often involves finding the optimal solution of a non-convex minimization problem

$$\min_{U \in \mathbb{R}^{n \times d}, V \in \mathbb{R}^{n \times d}} f(U, V) := \frac{1}{2} \|\mathcal{A}(UV^\top) - \boldsymbol{b}\|_F^2, \tag{1}$$

where $\text{rank}(M) = r \ll n$, of particular interest is the overparameterized case where $d > r$. The objective function $f(U, V)$ to be minimized is **non-convex**, **non-smooth**[3] and meanwhile **lacks**

---

[*]Part of this work was completed while Xixi Jia was a research fellow in the Department of Applied Mathematics at The Hong Kong Polytechnic University.

[†]Corresponding author.

[3]The non-smoothness pertains to $(U, V)$, as the magnitudes of these matrices can be highly unbalanced.

**coercivity**, presenting significant challenges in solving the optimization problem. Since the objective function exhibits certain key characteristics akin to the loss function of deep neural networks, problem (1) stands as a cornerstone in the study of more challenging non-convex problems, such as those encountered in deep learning [7–10]. For further discussions, please refer to [11].

Recent years have witnessed significant progress in the study of this non-convex optimization problem.

**(a) Progress on gradient descent algorithms.** Existing works such as [12–16] demonstrate that the non-convex objective function $f(U, V)$ possesses benign loss landscape, wherein all local minima are global, and concurrently, the Hessian exhibits negative eigenvalues at saddle points, allowing perturbed gradient descent algorithms to effectively escape them. To handle the non-smooth problem, prior studies [2, 12, 14, 17] introduce a regularization term $\frac{1}{8}\|U^\top U - V^\top V\|_F^2$ to the objective function. This regularization ensures balance between the norms of $U$ and $V$.

Very recently, Ye and Du [18] make a breakthrough on low-rank matrix factorization (LRMF), a specific setting of the problem (1) ($d = r, m \to \infty$), and prove that gradient descent, without perturbation and without the balance regularization on the objective function, converges at an R-linear rate to the global optimal solution of the non-convex problem from random initialization. Meanwhile, Stöger and Soltanolkotabi [19] study the global convergence of gradient descent for overparameterized ($d > r$) low-rank matrix sensing. However, overparameterization can significantly slow down gradient descent from achieving linear convergence to sub-linear rates, as analyzed in [15, 20]. Furthermore, Xiong et al. [20] proves that imbalanced initialization can expedite the convergence of gradient descent from sub-linear to linear rate. Nevertheless, gradient descent still requires a considerable amount of time to navigate away from saddle points, as discussed in Section 4. Additionally, the convergence rate of gradient descent is heavily reliant on the condition number of the matrix $M$, rendering it inefficient for solving ill-conditioned non-convex optimization problems.

**(b) Progress on advanced algorithms.** Given these deficiencies of first-order gradient methods, it is intriguing and crucial to investigate how computationally efficient higher-order algorithms, behave on this non-convex problem. Previously, Liu et al. [22] introduces a Gauss-Newton type method for symmetric LRMF (with $d = r$), and proved that Gauss-Newton method

Table 1: Comparisons of iteration complexity, with $\kappa$ as the condition number of the $n \times n$ matrix. "init." denotes initialization.

| Algorithm | init. | iteration complexity |
|---|---|---|
| GD [20] | random | $\kappa^{11}\log(\kappa^2/n) + \kappa^{10}\log(\kappa^6/\varepsilon)$ |
| PrecGD [15] | spectral | $\log(1/\varepsilon)$ |
| ScaledGD($\lambda$)[21] | random | $\log\kappa \cdot \log(\kappa n) + \log(1/\varepsilon)$ |
| AGN | random | $\log(1/\varepsilon)$ |

converges Q-linearly fast to a *critical point* of the non-convex optimization problem. Recently, Zilber and Nadle [23] prove that the Gauss-Newton method enjoys local quadratic convergence if the initialization lies within a small basin of attraction of the global optimal solution. All these works only guarantee local convergence and neglect the influence of saddle points on the convergence. *Global convergence of the Gauss-Newton method remains ambiguous.* Yue et al. [24] proves that the Newton method with cubic regularization converges quadratically fast from random initialization. However, their results are only applicable for symmetric MS with $d = r$, and the computational cost of Newton method in [24] for LRMS is very high. Lee and Stöger [25] prove that for rank-one matrix sensing, alternating least square method converges to the global optimal solution at a linear rate from random initialization. However, it is uncertain whether the results of [25] are applicable to the $r > 1$ case as well as the challenging overparameterized scenario ($d > r$).

Another line of work deals with the deficiencies of gradient descent by incorporating preconditioning matrices into the gradient direction, as demonstrated by [15, 26, 27, 21, 28]. Tanner and Wei [26] introduce a scaled alternating steepest descent method with diminishing step size and provides asymptotic convergence. Tong et al. [27] introduces ScaledGD and proves that, given a spectral initialization, ScaledGD converges at a linear rate to the global optimal solution of problem (1). Additionally, the convergence rate is independent of the condition number of $M$. The works in [15] and [21] focus on the symmetric matrix sensing, extending the ScaledGD to the overparameterized case by introducing a damping factor $\lambda$ to control the singularity of the preconditioning matrix. However, the damping factor $\lambda$ can decelerate the ScaledGD from escaping the saddle regions and the global iteration complexity becomes $\log\kappa \cdot \log(\kappa n) + \log(1/\varepsilon)$ as given in Table 1.

**Our contributions.** In this paper, we focus on the general model (1) which covers both symmetric matrix sensing ($M$ is symmetric and positive semi-definite) and asymmetric matrix sensing ($M$ is rectangular matrix), particularly with $d > r$. Building upon the insights from [23, 25, 26], we use an approximated Gauss-Newton (AGN) method for solving the non-convex optimization problem

(1). Notably, in each iteration, AGN performs computations akin to gradient descent, yet it exhibits a Q-linear convergence rate towards the global optimal solution from random initialization, with global iteration complexity $\log(1/\varepsilon)$ as shown in Table 1 AGN. Moreover, the Q-linear factor is independent of the condition number of $M$. Under certain conditions on the sensing operator $\mathcal{A}(\cdot)$, we can prove super-linear convergence of the AGN. This distinctive convergence property substantially enhances outcomes compared to existing methods in achieving global convergence, as shown in Table 1. Additionally, as a byproduct result, we establish that the prevailing preconditioned gradient descent methods are analogous to the Levenberg–Marquardt method within the Gauss-Newton framework.

To conclude, the contributions of this paper are as follows. First, we reformulate the symmetric and asymmetric matrix sensing in a unified way, based on which we design an approximated Gauss-Newton (AGN) method. We show that the existing preconditioned gradient descent algorithms, ScaledGD($\lambda$) and PrecGD in [15, 21] respectively, correspond to a certain type of the Gauss-Newton method. Then we analyze the behavior of GD, Scaled($\lambda$)/PrecGD and AGN at $\varepsilon$-neighborhood of the saddle points. The saddle point analysis shows that AGN is not attracted to saddle points, unlike GD and PrecGD. Specifically, GD and Scaled($\lambda$)/PrecGD achieve an objective function decrease of $o(\varepsilon)$, while AGN achieves a significantly larger, $\varepsilon$-independent decrease $\Theta(1)$. Finally, we prove the global Q-linear convergence of AGN for overparameterized non-convex LRMS. This significantly improves over algorithms like ScaledGD($\lambda$) and PrecGD, which achieve R-linear convergence but are hindered by saddle regions.

This paper is organized in the following ways. Section 2 provides an overview of related works. Section 3 outlines our AGN method for both symmetric MS and asymmetric MS, and explores the connections between AGN and Scaled($\lambda$)/PrecGD. Section 4 provides insights into the behavior of GD, ScaledGD, and AGN in the vicinity of saddle points. Section 5 outlines the main convergence result with a proof sketch, Section 6 presents experimental results, and Section 7 concludes the paper.

## 2 Related work

### 2.1 Overparameterization in matrix sensing and beyond

Overparameterization significantly impacts the optimization of both the non-convex low-rank matrix recovery problem [9, 15, 19, 29–31] and deep learning [32–35]. Specifically, [15, 29] analyzed that overparamterization can eliminate the spurious local minima of the non-convex low-rank matrix recovery problem. Meanwhile, since the exact rank parameter $r$ is not predetermined, in real-world applications one often relies on a moderately higher rank $d > r$, as elaborated in [21]. Geyer et al. [36] study the solution uniqueness in the overparamterized low-rank matrix sensing. Additional works examining overparameterized low-rank models include, but are not limited to [37–39].

### 2.2 Preconditioned/Scaled gradient descent

Preconditioned/Scaled gradient descent (ScaledGD) aims to enhance convergence by adjusting the gradient direction using preconditioning matrices, as specified in [15, 26, 27, 21, 28, 40]. It resolves the non-convex optimization problem (1) by the following iteration

$$\begin{cases} U_{t+1} = U_t - \eta \nabla_U f(U_t, V_t)(V_t^\top V_t)^{-1}, \\ V_{t+1} = V_t - \eta \nabla_V f(U_t, V_t)(U_t^\top U_t)^{-1}, \end{cases} \tag{2}$$

which corresponds to the methods in [27, 28, 40] for $r = d$. The author in [26] updates $U$ and $V$ in an alternating manner. If $U = V$ and $d > r$, then the iteration becomes ScaledGD($\lambda$)/PrecGD

$$U_{t+1} = U_t - \eta \nabla_U f(U_t, U_t)(U_t^\top U_t + \lambda_t I)^{-1}, \tag{3}$$

where $\lambda_t > 0$ can be either constant or time-varying, and the iteration corresponds to the methods in [15, 21]. ScaledGD($\lambda$) [21] and PrecGD [15] have been shown to achieve linear convergence to the global optimal solution, whether in a local context or in a global context. However, the parameter $\lambda_t$ can degrade the global convergence, as shown in Fig. (2) where PrecGD struggles to escape saddle regions before achieving local linear convergence.

### 2.3 Gauss-Newton method

The Gauss-Newton (GN) method is a widely recognized approach for nonlinear least-square

$$\min_{\boldsymbol{x} \in \mathbb{R}^n} \psi(\boldsymbol{x}) = \frac{1}{2} \|\phi(\boldsymbol{x})\|_2^2, \tag{4}$$

where $\phi : \mathbb{R}^n \to \mathbb{R}^m$ is a nonlinear, twice continuously differentiable function. In each iteration $t$, GN aims to solve least-squares of the linear approximation of $\phi(\boldsymbol{x})$ at point $\boldsymbol{x}_t$ [41]

$$\boldsymbol{x}_{t+1} = \boldsymbol{x}_t + \eta \Delta_t, \text{where } \Delta_t = \arg \min_{\Delta \in \mathbb{R}^n} \hat{\psi}(\boldsymbol{x}) = \frac{1}{2} \|\phi(\boldsymbol{x}_t) + J(\boldsymbol{x}_t)\Delta\|_2^2, \tag{5}$$

where $J(\boldsymbol{x}_t) = \phi'(\boldsymbol{x}_t)$ is the Jacobian of the nonlinear function $\phi(\cdot)$ at $\boldsymbol{x}_t$, and $\eta > 0$ is the step-length. $\eta = 1$ corresponds to the GN method, $\eta < 1$ corresponds to the damped Gauss-Newton method. The $\Delta_t$ is calculated by $\Delta_t = -[J(\boldsymbol{x}_t)^\top J(\boldsymbol{x}_t)]^{-1} J(\boldsymbol{x}_t)^\top \phi(\boldsymbol{x}_t)$, and $J(\boldsymbol{x}_t)^\top J(\boldsymbol{x}_t)$ is an approximation of the Hessian $H(\boldsymbol{x}_t) = J(\boldsymbol{x}_t)^\top J(\boldsymbol{x}_t) + \sum_{i=1}^n \phi_i(\boldsymbol{x}_t)\phi_i''(\boldsymbol{x}_t)$ if the second-order term is small. The GN method generally has a local convergence guarantee, indicating its effectiveness primarily within the vicinity of a solution [22, 23, 41]. When applied to the LRMS problem (1), we will show in Section 3.3 that the AGN method is closely related to the ScaledGD($\lambda$)/PrecGD method.

# 3 Proposed method

In this section, we begin by unifying the formulation of both symmetric matrix sensing (where $U = V$ and $M$ is positive semi-definite) and asymmetric matrix sensing problems. Based on the corresponding nonlinear least-squares problem, we introduce our approximated Gauss-Newton (AGN) method. Then we prove that the proposed AGN is a descent method. Furthermore, when addressing symmetric matrix sensing, we explore two distinct parameterization settings and demonstrate that employing an asymmetric parameterization can significantly enhance convergence. At last, we give some discussions on the relation between ScaledGD($\lambda$)/PrecGD and the proposed AGN method.

## 3.1 Approximated-Gauss-Newton (AGN) method for LRMS

We unify the formulation of symmetric and asymmetric LRMS into a single, simplified expression:

$$\min_{X \in \mathbb{R}^{2n \times d}} \psi(X) := \frac{1}{2} \|\mathcal{A}(PXX^\top Q) - \boldsymbol{b}\|_2^2, \tag{6}$$

where $P = [I \quad \mathbf{0}] \in \mathbb{R}^{n \times 2n}$, $Q = \begin{bmatrix} \mathbf{0} \\ I \end{bmatrix} \in \mathbb{R}^{2n \times n}$ and $I \in \mathbb{R}^{n \times n}$ is the identity matrix. The case $X = \begin{bmatrix} U \\ V \end{bmatrix}$ corresponds to asymmetric matrix sensing, $X = \begin{bmatrix} U \\ U \end{bmatrix}$ corresponds to symmetric matrix sensing with $U \in \mathbb{R}^{n \times n}$ and $V \in \mathbb{R}^{n \times n}$. The function $\psi(X)$ can be further rewritten as $\psi(X) = \frac{1}{2}\|\phi(X)\|_2^2$, where $\phi(X) = \mathcal{A}(PXX^\top Q) - \boldsymbol{b}$. For simplicity in notation, we denote $\mathcal{B}(X, Y) = \mathcal{A}(PXY^\top Q)$. By employing the Gauss-Newton framework as presented in Section 2.3, one can update the variable as $X_{t+1} = X_t + \eta \Delta(X_t)$ where

$$\Delta(X_t) = \arg \min_{\Delta \in \mathbb{R}^{2n \times d}} \frac{1}{2} \|\mathcal{B}(\Delta, X_t) + \mathcal{B}(X_t, \Delta) + \mathcal{B}(X_t, X_t) - \boldsymbol{b}\|_2^2. \tag{7}$$

However, as the Jacobian of the linear operator in the $l_2$-norm of Eq. (7) tends to be singular in our overparameterized setting ($d > r$), the Gauss-Newton method cannot be directly applied and one may consult for the Levenberg–Marquardt method. In this work, however, unlike the Levenberg–Marquardt method, we resolve the problem in Eq. (7) using the Gauss–Seidel method, whose advantages over the Levenberg–Marquardt method will be discussed in Section 3.3. Specifically, we update $X_t$ by

$$X_{t+\frac{1}{2}} = X_t + \eta \Delta(X_t), \ \Delta(X_t) = \arg \min_{\Delta \in \mathbb{R}^{2n \times d}} \frac{1}{2} \|\mathcal{B}(\Delta, X_t) + \mathcal{B}(X_t, X_t) - \boldsymbol{b}\|_2^2,$$

$$X_{t+1} = X_{t+\frac{1}{2}} + \eta \Delta(X_{t+\frac{1}{2}}), \ \Delta(X_{t+\frac{1}{2}}) = \arg \min_{\Delta \in \mathbb{R}^{2n \times d}} \frac{1}{2} \|\mathcal{B}(X_{t+\frac{1}{2}}, \Delta) + \mathcal{B}(X_{t+\frac{1}{2}}, X_{t+\frac{1}{2}}) - \boldsymbol{b}\|_2^2. \tag{8}$$

The sub-problems in Eq. (8) are quadratic minimization problems that can generally be solved very easily. In our matrix sensing problem, leveraging the RIP condition, we can approximate $\Delta(X_t)$ by

$$\hat{\Delta}(X_t) = \arg^+ \min_{\Delta \in \mathbb{R}^{2n \times d}} \frac{1}{2} \|\hat{\mathcal{B}}(\Delta, X_t) - \mathcal{A}^*(\mathcal{B}(X_t, X_t) - \boldsymbol{b})\|_F^2,$$

$$\hat{\Delta}(X_{t+\frac{1}{2}}) = \arg^+ \min_{\Delta \in \mathbb{R}^{2n \times d}} \frac{1}{2} \|\hat{\mathcal{B}}(X_{t+\frac{1}{2}}, \Delta) - \mathcal{A}^*(\mathcal{B}(X_{t+\frac{1}{2}}, X_{t+\frac{1}{2}}) - \boldsymbol{b})\|_F^2, \tag{9}$$

where $\hat{\mathcal{B}}(\Delta, X_t) = P\Delta X_t^\top Q$ and $\hat{\mathcal{B}}(X_{t+\frac{1}{2}}, \Delta) = PX_{t+\frac{1}{2}}\Delta^\top Q$, $\arg^+$ denotes the minimum norm solution as

$$
\begin{aligned}
\hat{\Delta}(X_t) &= P^\dagger \mathcal{A}^*(\mathcal{B}(X_t, X_t) - \boldsymbol{b}) Q^\top X_t (X_t^\top QQ^\top X_t^\top)^\dagger, \\
\hat{\Delta}(X_{t+\frac{1}{2}}) &= Q^{\top\dagger}[\mathcal{A}^*(\mathcal{B}(X_{t+\frac{1}{2}}, X_{t+\frac{1}{2}}) - \boldsymbol{b})]^\top PX_{t+\frac{1}{2}}(X_{t+\frac{1}{2}}^\top PP^\top X_{t+\frac{1}{2}})^\dagger,
\end{aligned}
\tag{10}
$$

which is a natural choice for degenerate least squares problem. $\dagger$ denotes the Moore-Penrose-Pseudo inverse, and $P^\dagger = P^\top$, $Q^{\top\dagger} = Q$ in our context. Then the AGN becomes [4]

$$
X_{t+\frac{1}{2}} = X_t - \eta\hat{\Delta}(X_t), \quad X_{t+1} = X_{t+\frac{1}{2}} - \eta\hat{\Delta}(X_{t+\frac{1}{2}}).
\tag{11}
$$

The specifics of the AGN method are presented in Algorithm 1 in Appendix. We demonstrate that the solution in Eq. (10) renders the AGN in Eq. (11) with a constant step-size $\eta > 0$ as a descent method.

**Lemma 1.** (Descent lemma) *For asymmetric matrix sensing, as long as $0 < \eta \leq 2/(1+\delta)$ and the Assumption 1 is satisfied, then there exists positive constant $\ell = (2\eta - (1+\delta)\eta^2)/2$ such that*

$$
\begin{aligned}
\psi(X_{t+\frac{1}{2}}) &\leq \psi(X_t) - \ell\|\hat{\mathcal{B}}(\hat{\Delta}(X_t), X_t)\|_F^2, \\
\psi(X_{t+1}) &\leq \psi(X_{t+\frac{1}{2}}) - \ell\|\hat{\mathcal{B}}(X_{t+\frac{1}{2}}, \hat{\Delta}(X_{t+\frac{1}{2}}))\|_F^2.
\end{aligned}
\tag{12}
$$

Lemma 1 suggests that AGN with a constant step-size is indeed a descent method for the overparameterized LRMS. In Section 5, we will prove that AGN converges globally at Q-linear rate.

The AGN method can be used not only for asymmetric MS but also for symmetric MS, as Eq. (6) offers a unified formulation for MS. While the application of AGN on symmetric MS will differ slightly from the asymmetric case.

## 3.2 Symmetric matrix sensing

Now we consider symmetric matrix sensing, which is a special case of model (6) and the matrix $M \in \mathbb{R}^{n \times n}$ is symmetric positive semi-definite (PSD). There are two different ways to deal with the symmetric case, depending on whether we constrain $X \in \mathcal{C}$ (symmetric parameterization), where $\mathcal{C} = \left\{ Z | Z = \begin{bmatrix} U \\ U \end{bmatrix}, U \in \mathbb{R}^{n \times d} \right\} \subset \mathbb{R}^{2n \times d}$ or $X \in \mathbb{R}^{2n \times d}$ (asymmetric parameterization).

**Setting 1: Symmetric parameterization.** In this case, the optimization variable $X \in \mathcal{C}$ such that $\mathcal{B}(X, X^\top) = \mathcal{A}(UU^\top)$ for matrix $U \in \mathbb{R}^{n \times d}$ and $\hat{\mathcal{B}}(\Delta, X_t) = [\hat{\mathcal{B}}(X_t, \Delta)]^\top$, thus the subproblems in Eq. (9) become

$$
\tilde{\Delta}(X_t) = \arg\min_{\Delta \in \mathcal{C}} \frac{1}{2}\|\hat{\mathcal{B}}(\Delta, X_t) - \mathcal{A}^*(\mathcal{B}(X_t, X_t) - \boldsymbol{b})\|_F^2,
\tag{13}
$$

where $X_t \in \mathcal{C}$. The optimal solution to the problem in Eq. (13) is provided by the following lemma.

**Lemma 2.** *Let $\hat{\Delta}(X_t)$ be the optimal solution of problem (9), then the optimal solution of problem (13) is $\tilde{\Delta}(X_t) = \begin{bmatrix} P\hat{\Delta}(X_t) \\ P\hat{\Delta}(X_t) \end{bmatrix}$.*

However, we find that if we constrain the search space to $\mathcal{C}$ and use the update $\tilde{\Delta}(X_t)$, the AGN with a constant step-size $\eta$ may not qualify as a descent method for our over-parameterized LRMS. Specifically, let $X_{t+1} = X_t - \eta\tilde{\Delta}(X_t)$, where $X_t, \tilde{\Delta}(X_t) \in \mathcal{C}$ and assume $X_t = \begin{bmatrix} U_t \\ U_t \end{bmatrix}$, we have

$$
\begin{aligned}
\psi(X_{t+1}) &= \frac{1}{2}\|\mathcal{B}(X_t, X_t) - \eta\mathcal{B}(X_t, \tilde{\Delta}(X_t)) - \eta\mathcal{B}(\tilde{\Delta}(X_t), X_t) + \eta^2\mathcal{B}(\tilde{\Delta}(X_t), \tilde{\Delta}(X_t)) - \boldsymbol{b}\|_2^2 \\
&= \psi(X_t) + \eta^2\|\mathcal{B}(X_t, \tilde{\Delta}(X_t))\|_2^2 - \eta\|\hat{\mathcal{B}}(X_t, \tilde{\Delta}(X_t))\|_F^2 + \frac{\eta^4}{2}\|\mathcal{B}(\tilde{\Delta}(X_t), \tilde{\Delta}(X_t))\|_2^2 \\
&\quad - \eta^3\left\langle \mathcal{B}(\tilde{\Delta}(X_t), X_t), \mathcal{B}(\tilde{\Delta}(X_t), \tilde{\Delta}(X_t)) \right\rangle + \eta^2\left\langle \mathcal{B}(X_t, X_t), \mathcal{B}(\tilde{\Delta}(X_t), \tilde{\Delta}(X_t)) \right\rangle.
\end{aligned}
\tag{14}
$$

Note that the term $\|\mathcal{B}(X_t, \tilde{\Delta}(X_t))\|_2^2 \leq (1+\delta)\|\mathcal{E}_t\|_F^2$, $\|\hat{\mathcal{B}}(X_t, \tilde{\Delta}(X_t))\|_F^2 \leq \|\mathcal{E}_t\|_F^2$ are all bounded and are closely related to $\psi(X_t)$, and $\mathcal{E}_t = \mathcal{A}^*(\mathcal{A}(U_t U_t^\top) - \boldsymbol{b})$. While the higher-order term w.r.t. $\tilde{\Delta}(X_t)$

$$
\begin{aligned}
\|\mathcal{B}(\tilde{\Delta}(X_t), \tilde{\Delta}(X_t))\|_2^2 &= \|\mathcal{A}(\mathcal{E}_t U_t (U_t^\top U_t)^{-2} U_t^\top \mathcal{E}_t^\top)\|_2^2 \\
&\geq (1-\delta)\|\mathcal{E}_t U_t (U_t^\top U_t)^{-\frac{1}{2}} (U_t^\top U_t)^{-1} (U_t^\top U_t)^{-\frac{1}{2}} U_t^\top \mathcal{E}_t^\top\|_F^2
\end{aligned}
\tag{15}
$$

can be extremely large such that $\psi(X_{t+1}) \geq \psi(X_t)$, as $\|\mathcal{E}_t U_t (U_t^\top U_t)^{-\frac{1}{2}}\|_F^2 \leq \|\mathcal{E}_t\|_F^2$ is bounded while $U_t^\top U_t$ tends to be singular in the over-parametrized case. Therefore, one cannot guarantee that the AGN method decreases with a constant step-size $\eta$, as illustrated by Fig. 3 $\text{AGN}_{\text{sym}}$ in Section 6. How should we approach the symmetric matrix sensing problem using AGN? One possible strategy is to relax the constraint $X \in \mathcal{C}$ to a larger search space $X \in \mathbb{R}^{2n \times d}$.

**Setting 2: Asymmetric parameterization.** Despite $M$ being a symmetric PSD matrix, one can still consider problem (6) with $X \in \mathbb{R}^{2n \times d}$ instead of $X \in \mathcal{C}$. Denote by $X^*$ and $X_c^*$ the optimal solution with $X^* \in \mathbb{R}^{2n \times d}$ and $X_c^* \in \mathcal{C}$ respectively, then it is easy to verify that $PX^*X^{*\top}Q = PX_c^*X_c^{*\top}Q$ for symmetric matrix sensing. Therefore, one can readily apply AGN using Algorithm 1 to solve problem (6) with $M$ being a symmetric PSD matrix. We will achieve the same convergence guarantee as in the case of asymmetric matrix sensing.

**Remark 1.** The symmetric MS discussed in setting 1 is a specific instance of problem (6), involving significantly fewer intrinsic variables compared to the asymmetric case discussed in setting 2, the search space $\mathcal{C}$ resides in a lower dimensional subspace of $\mathbb{R}^{2n \times d}$. Meanwhile, one can also explore the optimal solution for the symmetric MS within the expanded space $\mathbb{R}^{2n \times d}$ while maintaining the same minimum objective function value, as demonstrated in setting 2. These two approaches lead to significantly different optimization paths. From the above analysis, it is evident that different optimization paths demonstrate distinct decreasing properties in our over-parameterized LRMS problem. If we confine the optimization variable to $\mathcal{C}$, then the AGN with a constant step-size may not function as a descent method[5]. If we extend the optimization variable to the entire $\mathbb{R}^{2n \times d}$, we have a larger search space from which we can find a solution path that guarantees a significant decrease in the objective function. In Section 5, we will prove that in this case, AGN converges Q-linearly fast. These observations suggest that expanding the search space for a given optimization problem can lead to more efficient methods.

## 3.3 Comparisons with related works

Of particular relevance to this work are ScaledGD($\lambda$) [21] and PrecGD [15], which focus on over-parameterized symmetric low-rank matrix sensing. While these preconditioned gradient descent methods are not easily applicable to the general asymmetric matrix sensing problem, as we will discuss in the appendix Section A.1.3. In this section, we demonstrate that these preconditioned gradient descent methods are instances of the Levenberg–Marquardt method (specifically, the Gauss-Newton method for singular least square problems) applied to the symmetric low-rank matrix sensing problem. However, the Levenberg–Marquardt method is a more general approach than the preconditioned gradient descent method, particularly for nonlinear least square problems.

Since ScaledGD($\lambda$) [21] and PrecGD [15] consider over-parameterized symmetric matrix sensing, we constraint $X \in \mathcal{C}$ where $\mathcal{C}$ is defined in Section 3.2. It can be easily verified that the ScaledGD($\lambda$)/PrecGD for problem (6) corresponds to

$$
\begin{cases}
X_{t+1} = X_t - \eta \Delta(X_t, \lambda), \\
\hat{\Delta}(X_t, \lambda) = \arg\min_{\Delta \in \mathcal{C}} \frac{1}{2} \|\hat{\mathcal{B}}(\Delta, X_t) - \mathcal{A}^*(\mathcal{B}(X_t, X_t) - \boldsymbol{b})\|_F^2 + \lambda\|\Delta\|_F^2.
\end{cases}
\tag{16}
$$

It is apparent from Eq. (16) that $\lambda$ constrains the magnitude of the update $\Delta(X_t, \lambda)$ to be small compared to Eq. (13), thus ensuring that the preconditioned gradient descent method exhibits monotonically decreasing behavior, as analyzed in [42] and [21]. The Lemma 6 in [42] ensures that PrecGD is a descent method, as summarized by the following corollary

**Corollary 1.** *For symmetric matrix sensing, there exists positive constant $\ell_{X,\lambda}$ such that as long as $0 < \eta \le 2/\ell_{X,\lambda}$, the iteration given by Eq. (16) satisfies*

$$\psi(X_{t+1}) \le \psi(X_t) - \frac{\ell_{X,\lambda}}{2}\|\hat{\mathcal{B}}(\hat{\Delta}(X_t,\lambda), X_t)\|_F^2, \tag{17}$$

*where $\ell_{X,\lambda} = (1 + \delta)\left[4 + \frac{2\|\hat{\mathcal{B}}(X_t,X_t) - M\|_F + 4\|\hat{\mathcal{B}}(\hat{\Delta}(X_t,\lambda), X_t)\|_F}{\sigma_{\min}(X_t^\top X_t) + \lambda} + \left(\frac{\|\hat{\mathcal{B}}(\hat{\Delta}(X_t,\lambda), X_t)\|_F}{\sigma_{\min}(X_t^\top X_t) + \lambda}\right)^2\right]$.*

Similar to the Lemma 1 of our AGN method, the value $\|\hat{\mathcal{B}}(\hat{\Delta}(X_t,\lambda), X_t)\|_F^2$ plays very important role for the convergence of ScaledGD($\lambda$)/PrecGD. While we observe that the parameter $\lambda$ can notably impede the progress of ScaledGD($\lambda$)/PrecGD in escaping the saddle point, as illustrated in Fig. 2 and discussed in Section 6. In Section 4, we will prove that when $X_t$ is $\varepsilon$-close to the saddle points, $\|\hat{\mathcal{B}}(\hat{\Delta}(X_t,\lambda), X_t)\|_F^2$ will be as small as $o(\varepsilon)$, which explains why ScaledGD($\lambda$)/PrecGD converges slowly near saddle points. While in contrast, even if $X_t$ is $\varepsilon$-close to the saddle points, the value $\|\hat{\mathcal{B}}(\hat{\Delta}(X_t), X_t)\|_F^2$ in Eq. (12) is almost independent of $\varepsilon$, thus, saddle points cannot impede the convergence of the AGN method, which is also demonstrated in the left subfigure of Fig. 2.

# 4   Saddle point analysis on the population risk

Saddle points are special critical points in non-convex optimization problem, contributing significantly to the global convergence analysis of gradient-based algorithms in non-convex optimization. Past researches [18] has demonstrated that gradient descent may encounter difficulties in navigating away from saddle points, while our empirical findings in Section 5 demonstrate that the proposed AGN does not experience slowdowns caused by saddle points. Therefore, it's quite intriguing and crucial to understand the behaviors of gradient descent and AGN in the vicinity of saddle points. To this end, we study the saddle points of the population risk of the problem (6).

The population risk[6] associated with the objective function in Eq. (6) corresponds to the following non-convex matrix factorization problem:

$$\min_{X \in \mathbb{R}^{2n \times d}} \frac{1}{2}\|PXX^\top Q - M\|_2^2. \tag{18}$$

The objective function corresponds to $g(U,V) = \frac{1}{2}\|UV^\top - M\|_F^2, U \in \mathbb{R}^{n \times d}, V \in \mathbb{R}^{n \times d}$ which plays very important role in the saddle point analysis of Eq. (6). The saddle point of the non-convex objective $g(U,V)$ is denoted by $(U_s, V_s) \in \mathcal{S}$, where the set $\mathcal{S}$ is defined as follows:

$$\mathcal{S} = \left\{(U_s, V_s)|U_sV_s^\top = \Phi\mathcal{M}(\Sigma)\Psi^\top, M = \Phi\Sigma\Psi^\top, \mathcal{M} \in \mathfrak{M}/\mathfrak{I}\right\}, \tag{19}$$

where $M = \Phi\Sigma\Psi^\top$ is the SVD of the matrix $M$ and $\Phi \in \mathbb{R}^{n \times r}, \Psi \in \mathbb{R}^{n \times r}, \Sigma \in \mathbb{R}^{r \times r}$, $\mathfrak{M}$ is the set of mask operator[7] and $\mathfrak{I}$ is the identity operator.

Note that the gradient norm $\|\nabla g\|_F^2$ is intricately linked to the reduction of the objective function for gradient descent method. In our approach, the values $\|\hat{\mathcal{B}}(\hat{\Delta}(X_t), X_t)\|_F^2$ and $\|\hat{\mathcal{B}}(X_t, \hat{\Delta}(X_t))\|_F^2$ in Eq. (12), which correspond to $\|\nabla_U g(V^\top V)^{-\frac{1}{2}}\|_F^2$ and $\|\nabla_V g(U^\top U)^{-\frac{1}{2}}\|_F^2$ respectively, are directly tied to the reduction observed in the AGN method. Correspondingly for symmetric matrix sensing, the values $\|\hat{\mathcal{B}}(\hat{\Delta}(X_t,\lambda), X_t)\|_F^2$ in Eq. (17), which corresponds to $\|\nabla_U g(U^\top U + \lambda I)^{-\frac{1}{2}}\|_F^2$, is related to the reduction of the non-convex objective by SclaedGD [21] and PrecGD [15]. Now, we present the following theorem to describe the behavior of gradient descent, ScaledGD($\lambda$)/PrecGD and AGN in the vicinity of saddle points, by quantifying the values associated with their reductions in objective functions. For simplicity, we consider here $\text{rank}(M) = 1$.

**Theorem 1.** *Assume that $M$ is rank-1, the point $(\hat{U}, \hat{V})$ with $\hat{U} = U_s + \varepsilon N_u, \hat{V} = V_s + \varepsilon N_v$ is at the vicinity of the saddle point $(U_s, V_s) \in \mathcal{S}$ and $\varepsilon$ is sufficiently small, $N_u$ and $N_v$ are random Gaussian matrices that follow a standard normal distribution. Then with high probability we have the following results*

$$\text{(GD)} \qquad \|\nabla g\|_F^2 = o(\varepsilon)e_s + o(\varepsilon^2), \tag{20}$$

$$\text{(AGN)} \qquad \begin{cases} \|\nabla g_{\hat{U}}(\hat{V}^\top \hat{V})^{-\frac{1}{2}}\|_F^2 = \Theta(1)e_s + o(\varepsilon^2), \\ \|\nabla g_{\hat{V}}(\hat{U}^\top \hat{U})^{-\frac{1}{2}}\|_F^2 = \Theta(1)e_s + o(\varepsilon^2), \end{cases} \qquad (21)$$

*where $e_s = \|U_s V_s^\top - M\|_F^2$. Furthermore, by constraining $M$ to be positive semi-definite and $\hat{U} = \hat{V}, U_s = V_s$ (for symmetric matrix sensing), for bounded constant $c > 0$, we have*

$$\text{(ScaledGD($\lambda$))} \qquad \|\nabla g_{\hat{U}}(\hat{U}^\top \hat{U} + \lambda I)^{-\frac{1}{2}}\|_F^2 = \Theta\left(\frac{\varepsilon^2}{\varepsilon^2 + \lambda/c}\right)e_s + o(\varepsilon^2). \qquad (22)$$

Theorem 1 indicates that when the optimization variable $(\hat{U}, \hat{V})$ is $\varepsilon$-close to the saddle point of the non-convex objective function $g(\hat{U}, \hat{V})$, the norm of the gradient $\|\nabla g\|_F^2$ at $(\hat{U}, \hat{V})$ becomes as small as $o(\varepsilon)$. Hence, the convergence of gradient descent will be relatively slow, as shown in Fig. 2. Moreover, in Eq. (22), the value of $\lambda/c$ is typically much larger than $\varepsilon$. Consequently, the reduction achieved by ScaledGD($\lambda$)/PrecGD for symmetric matrix sensing is nearly identical to that of gradient descent $o(\varepsilon)$. In contrast, even if

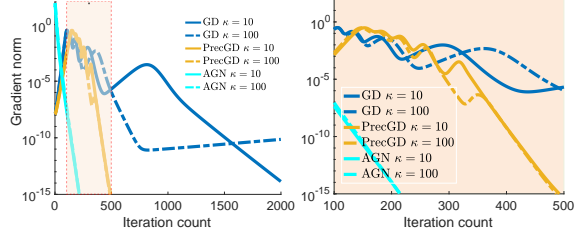

Figure 1: Illustration of the gradient norm for GD, PrecGD, and the proposed AGN, with the right subfigure showing a zoomed-in region of the left for iterations from 100 to 500.

$(\hat{U}, \hat{V})$ is $\varepsilon$-close to the saddle point, the reduction in the non-convex objective achieved by AGN in Eq. (21) is as significant as $\Theta(1)e_s + o(\varepsilon)$, where $\Theta(1)e_s$ is larger than $\varepsilon$ and is not dependent on $\varepsilon$, thus ensuring that AGN achieves a substantial decrease in the objective function near the saddle points. We also plot the gradient norm of GD, PrecGD and AGN in Fig. 1 for solving problem (6). It can be seen from Fig. 1 that the gradient norm of AGN decreases linearly to zero. While the gradient norm of PrecGD suffers from ups and downs before it is smaller than about $1 \times 10^{-7}$. This indicates that PrecGD is attracted to saddle points but can quickly escape, depending on the value of $\lambda$ as per Eq. (22). It becomes more challenging for GD to quickly escape all saddle points, which is due to Eq. (20). As shown in Fig. 1, GD's iterations encounter multiple saddle points before reaching the global minimum.

## 5 Global convergence analysis

We first recall the celebrated Restricted Isometry Property (RIP) [43], then we make some mild assumptions on the restricted isometry constant and the initialization of the variable $X_0$.

### 5.1 Assumptions and main result

**Assumption 1.** *The operator $\mathcal{A}(\cdot)$ satisfies the rank-$r+1$ RIP with constant $\delta_{r+1} := \delta$ for sufficiently small $\delta$.*

**Assumption 2.** *Let $X_0 \in \mathbb{R}^{2n \times d}$ be random Gaussian with elements sampled from $\mathcal{N}(0, \sigma)$ with $\sigma \le c_0 \|\Sigma\|_2/n$ for small $c_0$ and the step size $0 < \eta \le 2/(1 + \delta)$.*

Now we present our main result, which characterizes the global Q-linear convergence of the proposed AGN for the over-parameterized, non-convex, low-rank matrix sensing problem.

**Theorem 2** (Global Q-linear convergence). *Under the Assumption 1 and Assumption 2. Let $\psi^*$ be the global minimal value of $\psi(X)$ in Eq. (6) and $X_t, \forall t > 0$ is generated by Algorithm 1, then there exists constants $1 \ge \tau > 0$ such that*

$$\psi(X_{t+1}) - \psi^* \le c_q[\psi(X_t) - \psi^*], \forall t > 0, \qquad (23)$$

*where $c_q = (1 - \hat{\ell}\frac{1-\delta}{1+\delta}\tau) < 1$ and $\hat{\ell} = 2\eta - (1+\delta)\eta^2$. Meanwhile, if $\delta = 0, \eta = 1$, $c_q$ becomes 0.*

Theorem 2 echoes the observation in the left subfigure of Fig. 2 that AGN converges rapidly from random initialization and does not become trapped in saddle regions. The convergence result of AGN differs significantly from existing methods like ScaledGD($\lambda$) [21], PrecGD [15] and GD [19, 20], both empirically and theoretically. In the next section, we will present the sketch of our proof.

## 5.2 Proof sketch

The global Q-linear convergence of the AGN method relies on two conditions: **monotonically decreasing** (Lemma 1) and **decrease dominant** (Lemma 3). The monotonically decreasing condition ensures that the objective function decreases in each iteration, while the decrease dominant condition guarantees that the decrease in the function value is significantly larger than the distance between the current function value and the global minimum.

**Lemma 3** (Decrease dominant). *Under Assumption 1 and 2, let $X_t$ be updated by AGN method in Algorithm 1, then there exist $\tau_t^1, \tau_t^2$ and constant $\tau$ with $1 \geq \max\{\tau_t^1, \tau_t^2\} \geq \min\{\tau_t^1, \tau_t^2\} \geq \tau > 0$ and for constant $\delta > 0, 1 > \delta_c > 0$ such that*

$$
\begin{aligned}
\|\hat{\mathcal{B}}(\hat{\Delta}(X_t), X_t)\|_F^2 &\geq \frac{1 - \delta_c}{1 + \delta} \tau_t^1 [\psi(X_t) - \psi^*], \\
\|\hat{\mathcal{B}}(X_{t+\frac{1}{2}}, \hat{\Delta}(X_{t+\frac{1}{2}}))\|_F^2 &\geq \frac{1 - \delta_c}{1 + \delta} \tau_t^2 [\psi(X_{t+\frac{1}{2}}) - \psi^*],
\end{aligned}
\tag{24}
$$

*where $\hat{\Delta}(X_t)$ and $\hat{\Delta}(X_{t+\frac{1}{2}})$ is from Eq. (10).*

Note that the linear operator $\mathcal{A}(\cdot)$ and the $l_2$-norm is unitarily invariant, therefore for simplicity we consider $M$ to be a diagonal matrix $\Sigma \in \mathbb{R}^{n \times n}$ with $r$ nonzero elements on the diagonal. Specifically one can simply write $\Sigma = \Phi^\top M \Psi$ where $M = \Phi \Sigma \Psi^\top$ is the SVD of matrix $M$, as analyzed in [18] and [20]. Let $X_t = \begin{bmatrix} U_t \\ V_t \end{bmatrix}$, then $\|\hat{\mathcal{B}}(\hat{\Delta}(X_t), X_t)\|_F^2 = \|\mathcal{A}^*(\mathcal{A}(U_t V_t^\top - \Sigma))V_t V_t^\dagger\|_F^2$ and correspondingly $\psi(X_t) - \psi^* = \frac{1}{2}\|\mathcal{A}(U_t V_t^\top - \Sigma)\|_2^2$. According to the following Lemma 4 and the RIP condition in Definition 1, to guarantee the inequality in Eq. (24), we need to ensure that $\|(U_t V_t^\top - \Sigma)V_t V_t^\dagger\|_F^2 \geq \tau_1 \|U_t V_t^\top - \Sigma\|_F^2$ as presented by Lemma 5.

**Lemma 4.** *Assume that the operator $\mathcal{A}(\cdot)$ satisfies the RIP condition in Definition 1 and Assumption 1 with constant $\delta$, for any $U, V \in \mathbb{R}^{n \times d}$, $\Sigma \in \mathbb{R}^{n \times n}$ and $Z \in \mathbb{R}^{n \times n}$, then we have that*

$$
\|\mathcal{A}^*\mathcal{A}(UV^\top - \Sigma)Z\|_F \geq (1 - \delta_c)\|(UV^\top - \Sigma)Z\|_F
\tag{25}
$$

*for some $0 < \delta_c < 1$.*

**Lemma 5.** *Under Assumptions 1 and 2, if $X_t, \forall t > 0$ is generated by the AGN method in Algorithm 1 and let $X_t = \begin{bmatrix} U_t \\ V_t \end{bmatrix}$, then there exist constant $\tau, \tau_t^1, \tau_t^2$ with $\max\{\tau_t^1, \tau_t^2\} \geq \min\{\tau_t^1, \tau_t^2\} \geq \tau > 0$ such that*

$$
\begin{aligned}
\|(U_t V_t^\top - \Sigma)V_t V_t^\dagger\|_F^2 &\geq \tau_t^1 \|U_t V_t^\top - \Sigma\|_F^2, \\
\|(V_t U_{t+\frac{1}{2}}^\top - \Sigma)U_{t+\frac{1}{2}} U_{t+\frac{1}{2}}^\dagger\|_F^2 &\geq \tau_t^2 \|U_{t+\frac{1}{2}} V_t^\top - \Sigma\|_F^2.
\end{aligned}
\tag{26}
$$

Theorem 2 can be proven readily by combining all of these lemmas. Please refer to the Appendix for a more detailed proof.

## 6 Numerical experiments

In this section, we conduct experiments to demonstrate the effectiveness of the proposed AGN method for solving the over-parameterized non-convex matrix sensing problem. We set the ground truth matrix $M = U^*\Sigma V^{*\top}$, with $U^* \in \mathbb{R}^{n \times r}, V^* \in \mathbb{R}^{n \times r}$ random orthogonal matrices and $\Sigma$ is a diagonal matrix with condition number $\kappa$. We set $n = 100, r = 5, d = 3r$ and the number of sensing matrices $m = 50nr$. All experiments were conducted using MAT-LAB on a MacBook Pro with a 2.4 GHz quad-core Intel Core i5 CPU and 8 GB of memory.

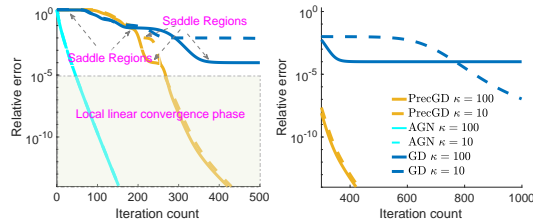

Figure 2: Comparison of convergence for PrecGD, GD, and AGN across various condition numbers, with the right subfigure extending the left by iterating from 300 to 1000.

**Comparison with representative methods.**
We compare AGN with GD [20] and PrecGD [15] on asymmetric over-parameterized matrix sensing.

All the competing methods are initialized with random Gaussian matrix with zero mean the variance $1/n$. We plot the training curves of the competing methods in Fig. 2, where the relative error is defined by $\|U_t V_t^\top - M^*\|_F / \|M^*\|_F$. GD's slow convergence is evident as it struggles to escape saddle points, as shown in the saddle regions of Fig. 2. Meanwhile, GD's final convergence rate depends on the condition number $\kappa$. Fig. 2 further shows that PrecGD's final convergence rate is independent of $\kappa$, though it still progresses slowly in saddle regions. In contrast, the relative error of AGN decreases rapidly, and its convergence remains unaffected by saddle points, consistent with Theorem 2.

Meanwhile, we compare the computational time of GD [20], PrecGD [15], ScaledGD($\lambda$) [21], and the proposed AGN on matrices of varying dimensions $n \times n$ under different condition number $\kappa$ in Table 2. It can be seen from Table 2 that AGN is significantly faster than the competing methods, particularly PrecGD and ScaledGD($\lambda$), while vanilla gradient descent converges much more slowly.

Table 2: Comparison of computational time for methods on matrices with varying dimensions $n$, measured in seconds. Here, $a+$ indicates time significantly exceeds $a$ seconds.

| Algorithm | $100 \times 100$ | | $500 \times 500$ | |
|---|---|---|---|---|
| | $\kappa = 10$ | $\kappa = 50$ | $\kappa = 10$ | $\kappa = 50$ |
| GD [20] | 500+ | 500+ | 5000+ | 5000+ |
| PrecGD [15] | 87.12 | 94.53 | 1979.45 | 1921.83 |
| ScaledGD($\lambda$)[21] | 55.87 | 69.07 | 1218.83 | 1258.71 |
| AGN(ours) | 27.24 | 25.52 | 617.58 | 632.24 |

**Asymmetric vs. symmetric parameterization of the symmetric MS.** We also conduct experiments to illustrate the differences between symmetric and asymmetric parameterization in symmetric matrix sensing, as discussed in subsection 3.2. As analyzed in section 3.2 setting 1, the matrix $U_t^\top U_t$ tends to be singular in over-parameterized matrix sensing, causing $\|\mathcal{B}(\tilde{\Delta}(X_t), \tilde{\Delta}(X_t))\|_2^2$ to become extremely large, leading to $\psi(X_{t+1}) \geq \psi(X_t)$, as shown in Eq. (14). Thus, we cannot guarantee that AGN in symmetric parameterization is a reliable descent method, as demonstrated by AGN$_{\text{sym}}$ in Fig. 3. However, with asymmetric parameterization, as outlined in subsection 3.2 setting 2, we can ensure the linear convergence of the AGN method, also illustrated by AGN$_{\text{asym}}$ in Fig. 3.

# 7 Conclusion

In this paper, we present an approximated Gauss-Newton (AGN) method for overparameterized non-convex low-rank matrix sensing problem. We demonstrate the close relationship between existing methods like ScaledGD($\lambda$) and PrecGD, and the Levenberg–Marquardt method, which is a variant of the Gauss-Newton method. Through saddle point analysis, we partially explain why gradient descent, Scaled($\lambda$)/PrecGD may be slowed down by saddle points, whereas the proposed AGN achieves fast convergence. Finally, we prove that the proposed AGN achieves Q-linear convergence from random Gauss initialization for the non-convex optimization problem. Our findings highlight the efficacy of (approximate) second-order methods in non-convex optimization, particularly for structured problems like non-convex matrix sensing. Moreover, our results can be extended to more complex challenges, such as optimizing deep neural networks.

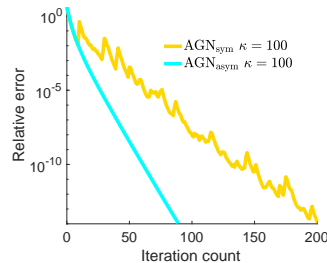

Figure 3: Convergence of AGN under Sym. and Asym. parameterization of symmetric LRMS.

**Limitations:** While the AGN method converges quickly and is efficient for low-rank matrix sensing problems, it may be less effective than gradient descent for general problems without low-rank structure due to the need to solve a least-squares problem for the Gauss-Newton direction. Future work will explore approximate methods, like the conjugate gradient, to address this. Additionally, our current saddle point analysis focuses on a simple non-convex LRMS case with a zero RIP constant, which we will generalize to more complex scenarios with larger RIP constants.

**Acknowledgements**

This research was supported by the National Key R&D Program of China (2022YFA1004100), the China NSFC projects under grant 62372359, 12226004, 62272375, the Research Center for Intelligent Operations Research and RGC Senior Research Fellow Scheme No. SRFS2223-5S02, and the Macao Science and Technology Development Fund under Grant 061/2020/A2.

## Footnotes

[4]The specific update is provided in Appendix Eq. (31), Eq. (32) and Eq. (33).

[5]One can employ a line search algorithm to ensure the decrease in the objective function value. Nevertheless, line search will slow-down the convergence and is not the primary focus of this paper.

[6]The population risk of problem (6) corresponds to $m \to \infty$ in Eq. (1) and Eq. (6).

[7]$\mathcal{M}(\cdot) : \mathbb{R}^{r \times r} \to \mathbb{R}^{r \times r}$ is a mask operator which maps the element $X_{i,j}$ of its input $X$ into zero or $X_{i,j}$.

# References

[1] Chih-Fan Chen, Chia-Po Wei, and Yu-Chiang Frank Wang. Low-rank matrix recovery with structural incoherence for robust face recognition. In *2012 IEEE conference on computer vision and pattern recognition*, pages 2618–2625. IEEE, 2012.

[2] Dohyung Park, Anastasios Kyrillidis, Constantine Carmanis, and Sujay Sanghavi. Non-square matrix sensing without spurious local minima via the Burer-Monteiro approach. In *Artificial Intelligence and Statistics*, pages 65–74. PMLR, 2017.

[3] Emmanuel J Candes. The restricted isometry property and its implications for compressed sensing. *Comptes rendus. Mathematique*, 346(9-10):589–592, 2008.

[4] Shuangjiang Li and Hairong Qi. A Douglas-Rachford splitting approach to compressed sensing image recovery using low-rank regularization. *IEEE Transactions on Image Processing*, 24(11): 4240–4249, 2015.

[5] Yuejie Chi, Yue M Lu, and Yuxin Chen. Nonconvex optimization meets low-rank matrix factorization: An overview. *IEEE Transactions on Signal Processing*, 67(20):5239–5269, 2019.

[6] Vasileios Charisopoulos, Yudong Chen, Damek Davis, Mateo Díaz, Lijun Ding, and Dmitriy Drusvyatskiy. Low-rank matrix recovery with composite optimization: good conditioning and rapid convergence. *Foundations of Computational Mathematics*, 21(6):1505–1593, 2021.

[7] Sanjeev Arora, Nadav Cohen, Wei Hu, and Yuping Luo. Implicit regularization in deep matrix factorization. In *Advances in Neural Information Processing Systems*, pages 7413–7424, 2019.

[8] Jikai Jin, Zhiyuan Li, Kaifeng Lyu, Simon Shaolei Du, and Jason D Lee. Understanding incremental learning of gradient descent: A fine-grained analysis of matrix sensing. In *International Conference on Machine Learning*, pages 15200–15238. PMLR, 2023.

[9] Yuanzhi Li, Tengyu Ma, and Hongyang Zhang. Algorithmic regularization in over-parameterized matrix sensing and neural networks with quadratic activations. In *Conference On Learning Theory*, pages 2–47. PMLR, 2018.

[10] Sanjeev Arora, Simon Du, Wei Hu, Zhiyuan Li, and Ruosong Wang. Fine-grained analysis of optimization and generalization for overparameterized two-layer neural networks. In *International Conference on Machine Learning*, pages 322–332. PMLR, 2019.

[11] Simon S Du, Wei Hu, and Jason D Lee. Algorithmic regularization in learning deep homogeneous models: Layers are automatically balanced. In *Advances in neural information processing systems*, pages 382–393, 2018.

[12] Rong Ge, Chi Jin, and Yi Zheng. No spurious local minima in nonconvex low rank problems: A unified geometric analysis. In *International Conference on Machine Learning*, pages 1233–1242. PMLR, 2017.

[13] Zhihui Zhu, Qiuwei Li, Gongguo Tang, and Michael B Wakin. The global optimization geometry of low-rank matrix optimization. *IEEE Transactions on Information Theory*, 67(2):1308–1331, 2021.

[14] Xingguo Li, Junwei Lu, Raman Arora, Jarvis Haupt, Han Liu, Zhaoran Wang, and Tuo Zhao. Symmetry, saddle points, and global optimization landscape of nonconvex matrix factorization. *IEEE Transactions on Information Theory*, 65(6):3489–3514, 2019.

[15] Jialun Zhang, Salar Fattahi, and Richard Y Zhang. Preconditioned gradient descent for over-parameterized nonconvex matrix factorization. *Advances in Neural Information Processing Systems*, 34:5985–5996, 2021.

[16] Srinadh Bhojanapalli, Behnam Neyshabur, and Nathan Srebro. Global optimality of local search for low rank matrix recovery. In *Advances in Neural Information Processing Systems*, pages 3880–3888, 2016.

[17] Stephen Tu, Ross Boczar, Max Simchowitz, Mahdi Soltanolkotabi, and Ben Recht. Low-rank solutions of linear matrix equations via procrustes flow. In *International Conference on Machine Learning*, pages 964–973. PMLR, 2016.

[18] Tian Ye and Simon S Du. Global convergence of gradient descent for asymmetric low-rank matrix factorization. In *Advances in Neural Information Processing Systems*, pages 1429–1439, 2021.

[19] Dominik Stöger and Mahdi Soltanolkotabi. Small random initialization is akin to spectral learning: Optimization and generalization guarantees for overparameterized low-rank matrix reconstruction. *Advances in Neural Information Processing Systems*, 34:23831–23843, 2021.

[20] Nuoya Xiong, Lijun Ding, and Simon Shaolei Du. How over-parameterization slows down gradient descent in matrix sensing: The curses of symmetry and initialization. In *The Twelfth International Conference on Learning Representations*, 2023.

[21] Xingyu Xu, Yandi Shen, Yuejie Chi, and Cong Ma. The power of preconditioning in overparameterized low-rank matrix sensing. In *International Conference on Machine Learning*, pages 38611–38654. PMLR, 2023.

[22] Xin Liu, Zaiwen Wen, and Yin Zhang. An efficient Gauss-Newton algorithm for symmetric low-rank product matrix approximations. *SIAM Journal on Optimization*, 25(3):1571–1608, 2015.

[23] Pini Zilber and Boaz Nadler. Gnmr: A provable one-line algorithm for low rank matrix recovery. *SIAM journal on mathematics of data science*, 4(2):909–934, 2022.

[24] Man-Chung Yue, Zirui Zhou, and Anthony Man-Cho So. On the quadratic convergence of the cubic regularization method under a local error bound condition. *SIAM Journal on Optimization*, 29(1):904–932, 2019.

[25] Kiryung Lee and Dominik Stöger. Randomly initialized alternating least squares: Fast convergence for matrix sensing. *SIAM Journal on Mathematics of Data Science*, 5(3):774–799, 2023.

[26] Jared Tanner and Ke Wei. Low rank matrix completion by alternating steepest descent methods. *Applied and Computational Harmonic Analysis*, 40(2):417–429, 2016.

[27] Tian Tong, Cong Ma, and Yuejie Chi. Accelerating ill-conditioned low-rank matrix estimation via scaled gradient descent. *Journal of Machine Learning Research*, 22(150):1–63, 2021.

[28] Xixi Jia, Hailin Wang, Jiangjun Peng, Xiangchu Feng, and Deyu Meng. Preconditioning matters: Fast global convergence of non-convex matrix factorization via scaled gradient descent. In *Advances in Neural Information Processing Systems*, volume 36, 2023.

[29] Richard Y Zhang. Sharp global guarantees for nonconvex low-rank matrix recovery in the overparameterized regime. *arXiv preprint arXiv:2104.10790*, 2021.

[30] Jianhao Ma and Salar Fattahi. Global convergence of sub-gradient method for robust matrix recovery: Small initialization, noisy measurements, and over-parameterization. *Journal of Machine Learning Research*, 24(96):1–84, 2023.

[31] Lijun Ding, Liwei Jiang, Yudong Chen, Qing Qu, and Zhihui Zhu. Rank overspecified robust matrix recovery: Subgradient method and exact recovery. *Advances in Neural Information Processing Systems*, 34:26767–26778, 2021.

[32] Zeyuan Allen-Zhu, Yuanzhi Li, and Yingyu Liang. Learning and generalization in overparameterized neural networks, going beyond two layers. In *Advances in Neural Information Processing Systems*, pages 6158–6169, 2019.

[33] Cong Fang, Jason Lee, Pengkun Yang, and Tong Zhang. Modeling from features: a mean-field framework for over-parameterized deep neural networks. In *Conference on learning theory*, pages 1887–1936. PMLR, 2021.

[34] Itay Safran and Ohad Shamir. Spurious local minima are common in two-layer relu neural networks. In *International conference on machine learning*, pages 4433–4441. PMLR, 2018.

[35] Difan Zou, Yuan Cao, Dongruo Zhou, and Quanquan Gu. Gradient descent optimizes over-parameterized deep relu networks. *Machine learning*, 109:467–492, 2020.

[36] Kelly Geyer, Anastasios Kyrillidis, and Amir Kalev. Low-rank regularization and solution uniqueness in over-parameterized matrix sensing. In *International Conference on Artificial Intelligence and Statistics*, pages 930–940. PMLR, 2020.

[37] Samet Oymak and Mahdi Soltanolkotabi. Overparameterized nonlinear learning: Gradient descent takes the shortest path? In *International Conference on Machine Learning*, pages 4951–4960. PMLR, 2019.

[38] Mahdi Soltanolkotabi, Adel Javanmard, and Jason D Lee. Theoretical insights into the optimization landscape of over-parameterized shallow neural networks. *IEEE Transactions on Information Theory*, 65(2):742–769, 2018.

[39] Richard Y Zhang. Improved global guarantees for the nonconvex burer–monteiro factorization via rank overparameterization. *arXiv preprint arXiv:2207.01789*, 2022.

[40] K Adithya Apuroop. A riemannian geometry for low-rank matrix completion. *arXiv preprint arXiv:1211.1550*, 2012.

[41] Serge Gratton, Amos S Lawless, and Nancy K Nichols. Approximate Gauss-Newton methods for nonlinear least squares problems. *SIAM Journal on Optimization*, 18(1):106–132, 2007.

[42] Gavin Zhang, Salar Fattahi, and Richard Y Zhang. Preconditioned gradient descent for over-parameterized nonconvex burer–monteiro factorization with global optimality certification. *Journal of Machine Learning Research*, 24(163):1–55, 2023.

[43] Benjamin Recht, Maryam Fazel, and Pablo A Parrilo. Guaranteed minimum-rank solutions of linear matrix equations via nuclear norm minimization. *SIAM review*, 52(3):471–501, 2010.

[44] Terence Tao. *Topics in random matrix theory*, volume 132. American Mathematical Soc., 2012.

# A    Appendix / supplemental material

## A.1    Preliminaries and more details

### A.1.1    The definition of RIP

**Definition 1** (Restricted Isometry Property). *The linear operator $\mathcal{A}(\cdot)$ is said to satisfy rank-$r$ RIP with a constant $\delta_r \in [0, 1)$ if for all matrices $M$ of rank at most $r$ the following condition holds*

$$(1 - \delta_r)\|M\|_F^2 \leq \|\mathcal{A}(M)\|_2^2 \leq (1 + \delta_r)\|M\|_F^2. \tag{27}$$

### A.1.2    The main AGN algorithm

---
**Algorithm 1:** AGN for matrix sensing

---
**Data:** $\mathcal{A}(\cdot)$, $\boldsymbol{b}$, $\eta$ and the random Gauss initialization $X_0$, $t = 0$.
**Result:** The estimated solution $X$ and the low-rank matrix $\hat{M} = PXX^\top Q$.
**while** *not end* **do**
    $t = t + 1$;
    Update the approximated Gauss-Newton direction by Eq. (10);
    Update $X_t$ and $X_{t+\frac{1}{2}}$ by Eq. (11);
**end**

---

### A.1.3    ScaledGD($\lambda$)/PrecGD for LRMS problem (6)

Directly applying the ScaledGD($\lambda$)/PrecGD methods in [21] and [15] respectively to problem (6) will lead to the following iterative update as

$$X_{k+1} = X_k - \eta \nabla \psi(X_t)(X_t^\top X_t + \lambda_t I)^{-1}, \tag{28}$$

where $\lambda_t$ can be either constant or time-varying. Note that in our LRMS problem (6), $X_t = \begin{bmatrix} U_t \\ V_t \end{bmatrix}$, therefore the update in Eq. (28) becomes

$$U_{t+1} = U_t - \eta \mathcal{A}^* \mathcal{A}(U_t V_t^\top - \Sigma)V_t(U_t^\top U_t + V_t^\top V_t + \lambda_t I)^{-1}, \tag{29}$$

and

$$V_{t+1} = V_t - \eta[\mathcal{A}^* \mathcal{A}(U_t V_t^\top - \Sigma)]^\top U_t(U_t^\top U_t + V_t^\top V_t + \lambda_t I)^{-1}, \tag{30}$$

which leads to quite different method compared to our AGN with iterations in $U_t$ and $V_t$ given by the following Eq. (32) and Eq. (33). Meanwhile, as shown in Fig. 4, using Eq. (28) for our LRMS (denoted by PrecGD) results in slower convergence compared to our AGN method, and moreover the convergence rate is dependent on the condition number of $M$ as $\kappa(M)$. In contrast, the proposed AGN converges very quickly, and its convergence rate is independent of the condition number $\kappa(M)$.

### A.1.4    Some detailed derivations.

Before presenting the proof, we first provide additional details regarding the update described in Eq. (11). Specifically, let $X_t = \begin{bmatrix} U_t \\ V_t \end{bmatrix}$, with $U_t = \begin{bmatrix} \hat{U}_t \\ J_t \end{bmatrix}$ and $V_t = \begin{bmatrix} \hat{V}_t \\ K_t \end{bmatrix}$ where $\hat{U}_t, \hat{V}_t \in \mathbb{R}^{r \times d}$ and $J_t, K_t \in \mathbb{R}^{(n-r) \times d}$. Since $\Sigma = \begin{bmatrix} \Sigma_r & 0 \\ 0 & 0 \end{bmatrix}$ with $\Sigma_r \in \mathbb{R}^{r \times r}$, Thus, the update in Eq. (11) can be succinctly expressed as

$$X_{t+\frac{1}{2}} = X_t - \eta \hat{\Delta}(X_t) = \begin{bmatrix} U_{t+1} \\ V_t \end{bmatrix}, X_{t+1} = X_{t+\frac{1}{2}} - \eta \hat{\Delta}(X_{t+\frac{1}{2}}) = \begin{bmatrix} U_{t+1} \\ V_{t+1} \end{bmatrix}, \tag{31}$$

with $U_{t+1}$ and $V_{t+1}$ given by

$$U_{t+1} = U_t - \eta \mathcal{A}^* \mathcal{A}(U_t V_t^\top - \Sigma)V_t(V_t^\top V_t)^\dagger, \tag{32}$$

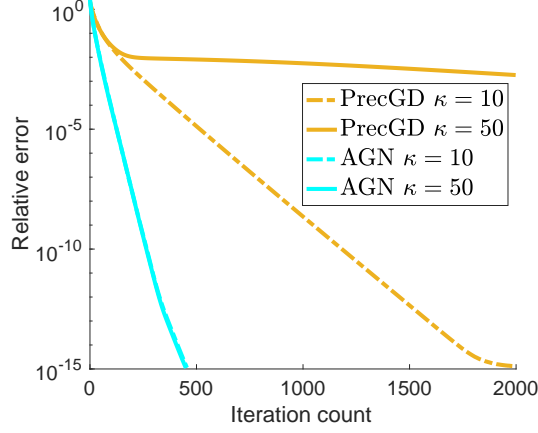

Figure 4: Convergence of AGN and the method by using Eq. (28) (denoted by PrecGD) for LRMS with different condition numbers on matrix $M$.

and

$$V_{t+1} = V_t - \eta[\mathcal{A}^*\mathcal{A}(U_{t+1}V_t^\top - \Sigma)]^\top U_{t+1}(U_{t+1}^\top U_{t+1})^\dagger, \tag{33}$$

and correspondingly

$$\hat{\Delta}(X_t) = \begin{bmatrix} \mathcal{A}^*\mathcal{A}(U_tV_t^\top - \Sigma)V_t(V_t^\top V_t)^\dagger \\ \mathbf{0} \end{bmatrix}, \tag{34}$$

$$\hat{\Delta}(X_{t+\frac{1}{2}}) = \begin{bmatrix} \mathbf{0} \\ [\mathcal{A}^*\mathcal{A}(U_{t+1}V_t^\top - \Sigma)]^\top U_{t+1}(U_{t+1}^\top U_{t+1})^\dagger \end{bmatrix}. \tag{35}$$

In consequence, we have

$$\|\hat{\mathcal{B}}(\hat{\Delta}(X_t), X_t)\|_F^2 = \mathcal{A}^*\mathcal{A}(U_tV_t^\top - \Sigma)V_t(V_t^\top V_t)^\dagger V_t^\top, \tag{36}$$

and

$$\|\hat{\mathcal{B}}(X_{t+\frac{1}{2}}, \hat{\Delta}(X_{t+\frac{1}{2}}))\|_F^2 = U_{t+1}(U_{t+1}^\top U_{t+1})^\dagger U_{t+1}^\top \mathcal{A}^*\mathcal{A}(U_{t+1}V_t^\top - \Sigma). \tag{37}$$

Further more, we can reformulate $U_{t+1}$ and $V_{t+1}$ as

$$\begin{aligned} U_{t+1} &= U_t - \eta\mathcal{A}^*\mathcal{A}(U_tV_t^\top - \Sigma)V_t(V_t^\top V_t)^\dagger \\ &= U_t - \eta(U_tV_t^\top - \Sigma)V_t(V_t^\top V_t)^\dagger + \eta(I - \mathcal{A}^*\mathcal{A})(U_tV_t^\top - \Sigma)V_t(V_t^\top V_t)^\dagger \\ &= (1-\eta)\begin{bmatrix}\hat{U}_t \\ J_t\end{bmatrix} + \eta\begin{bmatrix}\Sigma_r & 0 \\ 0 & 0\end{bmatrix}\begin{bmatrix}\hat{V}_t \\ K_t\end{bmatrix}\left(\hat{V}_t^\top\hat{V}_t + K_t^\top K_t\right)^\dagger \\ &\quad + \eta(I - \mathcal{A}^*\mathcal{A})\left(\begin{bmatrix}\hat{U}_t\hat{V}_t^\top - \Sigma_r & \hat{U}_tK_t^\top \\ J_t\hat{V}_t^\top & J_tK_t^\top\end{bmatrix}\right)\begin{bmatrix}\hat{V}_t \\ K_t\end{bmatrix}\left(\hat{V}_t^\top\hat{V}_t + K_t^\top K_t\right)^\dagger \\ &= (1-\eta)\begin{bmatrix}\hat{U}_t \\ J_t\end{bmatrix} + \eta\begin{bmatrix}\Sigma_r\hat{V}_t\left(\hat{V}_t^\top\hat{V}_t + K_t^\top K_t\right)^\dagger \\ 0\end{bmatrix} \\ &\quad + \eta(I - \mathcal{A}^*\mathcal{A})\left(\begin{bmatrix}\hat{U}_t\hat{V}_t^\top - \Sigma_r & \hat{U}_tK_t^\top \\ J_t\hat{V}_t^\top & J_tK_t^\top\end{bmatrix}\right)\begin{bmatrix}\hat{V}_t \\ K_t\end{bmatrix}\left(\hat{V}_t^\top\hat{V}_t + K_t^\top K_t\right)^\dagger, \end{aligned} \tag{38}$$

and

$$\begin{aligned} V_{t+1} &= V_t - \eta[\mathcal{A}^*\mathcal{A}(U_{t+1}V_t^\top - \Sigma)]^\top U_{t+1}(U_{t+1}^\top U_{t+1})^\dagger \\ &= (1-\eta)\begin{bmatrix}\hat{V}_t \\ K_t\end{bmatrix} + \eta\begin{bmatrix}\Sigma_r\hat{U}_{t+1}\left(\hat{U}_{t+1}^\top\hat{U}_{t+1} + J_{t+1}^\top J_{t+1}\right)^\dagger \\ 0\end{bmatrix} \\ &\quad + \eta(I - \mathcal{A}^*\mathcal{A})\begin{bmatrix}\hat{V}_t\hat{U}_{t+1}^\top - \Sigma_r & \hat{V}_tJ_{t+1}\hat{U}_{t+1} \\ K_t\hat{U}_{t+1}^\top & K_tJ_{t+1}^\top\end{bmatrix}\begin{bmatrix}\hat{U}_{t+1} \\ J_{t+1}\end{bmatrix}\left(\hat{U}_{t+1}^\top\hat{U}_{t+1} + J_{t+1}^\top J_{t+1}\right)^\dagger. \end{aligned} \tag{39}$$

We denote

$$\begin{bmatrix} E_{U_t} \\ E_{J_t} \end{bmatrix} = (I - \mathcal{A}^* \mathcal{A}) \left( \begin{bmatrix} \hat{U}_t \hat{V}_t^\top - \Sigma_r & \hat{U}_t K_t^\top \\ J_t \hat{V}_t^\top & J_t K_t^\top \end{bmatrix} \right) \begin{bmatrix} \hat{V}_t \\ K_t \end{bmatrix} \left( \hat{V}_t^\top \hat{V}_t + K_t^\top K_t \right)^\dagger, \tag{40}$$

and

$$\begin{bmatrix} E_{V_t} \\ E_{K_t} \end{bmatrix} = (I - \mathcal{A}^* \mathcal{A}) \begin{bmatrix} \hat{V}_t \hat{U}_{t+1}^\top - \Sigma_r & \hat{V}_t J_{t+1} \hat{U}_{t+1} \\ K_t \hat{U}_{t+1}^\top & K_t J_{t+1}^\top \end{bmatrix} \begin{bmatrix} \hat{U}_{t+1} \\ J_{t+1} \end{bmatrix} \left( \hat{U}_{t+1}^\top \hat{U}_{t+1} + J_{t+1}^\top J_{t+1} \right)^\dagger. \tag{41}$$

Then it follows

$$\hat{U}_{t+1} = (1 - \eta)\hat{U}_t + \eta \Sigma_r \hat{V}_t \left( \hat{V}_t^\top \hat{V}_t + K_t^\top K_t \right)^\dagger + \eta E_{U_t}, \tag{42}$$

similarly

$$\hat{V}_{t+1} = (1 - \eta)\hat{V}_t + \eta \Sigma_r \hat{U}_{t+1}^\top \left( \hat{U}_{t+1}^\top \hat{U}_{t+1} + J_{t+1}^\top J_{t+1} \right)^\dagger + \eta E_{V_t}, \tag{43}$$

and

$$\begin{aligned} J_{t+1} &= (1 - \eta)J_t + \eta E_{J_t}, \\ K_{t+1} &= (1 - \eta)K_t + \eta E_{K_t}. \end{aligned} \tag{44}$$

Now we give an estimation of the norm of $E_{U_t}$, $E_{V_t}$, $E_{J_t}$ and $E_{K_t}$. Note that

$$\begin{aligned} \mathcal{A}^* \mathcal{A}(U_t V_t^\top - \Sigma)V_t(V_t^\top V_t)^\dagger &= \sum_{i=1}^m \left\langle A_i, U_t V_t^\top - \Sigma \right\rangle A_i V_t(V_t^\top V_t)^\dagger \\ &= \sum_{i=1}^m \left\langle \begin{bmatrix} A_i^1 \\ A_i^2 \end{bmatrix}, \begin{bmatrix} \hat{U}_t \hat{V}_t^\top - \Sigma_r & \hat{U}_t K_t^\top \\ J_t \hat{V}_t^\top & J_t K_t^\top \end{bmatrix} \right\rangle \begin{bmatrix} A_i^1 \\ A_i^2 \end{bmatrix} V_t(V_t^\top V_t)^\dagger \end{aligned} \tag{45}$$

where $A_i = \begin{bmatrix} A_i^1 \\ A_i^2 \end{bmatrix}$ and $A_i^1 \in \mathbb{R}^{r \times n}$, $A_i^2 \in \mathbb{R}^{(n-r) \times n}$ for all $i = 1, \cdots, m$. Thus we have

$$E_{U_t} = (I - \mathcal{A}_1^* \mathcal{A}_1) \left( \begin{bmatrix} \hat{U}_t \hat{V}_t^\top - \Sigma_r & \hat{U}_t K_t^\top \end{bmatrix} \right) V_t(V_t^\top V_t)^\dagger + \mathcal{A}_1^* \mathcal{A}_2 (J_t V_t^\top) V_t(V_t^\top V_t)^\dagger \tag{46}$$

where $\mathcal{A}_1^* \mathcal{A}_1(\cdot) = \sum_{i=1}^m \left\langle A_i^1, \cdot \right\rangle A_i^1$ and $\mathcal{A}_1^* \mathcal{A}_2(\cdot) = \sum_{i=1}^m \left\langle A_i^2, \cdot \right\rangle A_i^1$. Note that

$$\| (I - \mathcal{A}_1^* \mathcal{A}_1) \left( \begin{bmatrix} \hat{U}_t \hat{V}_t^\top - \Sigma_r & \hat{U}_t K_t^\top \end{bmatrix} \right) V_t(V_t^\top V_t)^\dagger \|_2 \leq c_{\delta_1} e_t t = O(c_e^{-t} t) \tag{47}$$

and for sufficiently small RIP constant $\delta$ in Assumption 1, we can assume that

$$\| \mathcal{A}_1^* \mathcal{A}_2 (J_t V_t^\top) V_t(V_t^\top V_t)^\dagger \|_2 \leq c_{\delta_2} \|J_t\|_2 \tag{48}$$

where $c_e > 1$ and $c_{\delta_2}$ is constant, thus we know that the upper bound of $\|E_{U_t}\|_2$ is monotonically decreasing. The same results hold for $E_{V_t}$.

Meanwhile, we have

$$E_{J_t} = (I - \mathcal{A}_2^* \mathcal{A}_2)(J_t V_t^\top) V_t(V_t^\top V_t)^\dagger + \mathcal{A}_1^* \mathcal{A}_2 \left( \begin{bmatrix} \hat{U}_t \hat{V}_t^\top - \Sigma_r & \hat{U}_t K_t^\top \end{bmatrix} \right) V_t(V_t^\top V_t)^\dagger \tag{49}$$

where $\mathcal{A}_2^* \mathcal{A}_2(\cdot) = \sum_{i=1}^m \left\langle A_i^2, \cdot \right\rangle A_i^2$ and $\mathcal{A}_2^* \mathcal{A}_1(\cdot) = \sum_{i=1}^m \left\langle A_i^1, \cdot \right\rangle A_i^2$.

Under the Assumption 1, for sufficiently small $\delta$ and the RIP condition, we can assume that

$$\| (I - \mathcal{A}_2^* \mathcal{A}_2)(J_t V_t^\top) V_t(V_t^\top V_t)^\dagger \|_F \leq c_{\delta_1} \|J_t\|_F$$

for $c_{\delta_1} \leq 1$. Meanwhile we have

$$\begin{aligned} \| \mathcal{A}_1^* \mathcal{A}_2 \left( \begin{bmatrix} \hat{U}_t \hat{V}_t^\top - \Sigma_r & \hat{U}_t K_t^\top \end{bmatrix} \right) V_t(V_t^\top V_t)^\dagger \|_F &\leq t c_J \| \mathcal{A}_1^* \mathcal{A}_2 \left( \begin{bmatrix} \hat{U}_t \hat{V}_t^\top - \Sigma_r & \hat{U}_t K_t^\top \end{bmatrix} \right) \|_F \\ &\leq c_{\delta_2} e_t t. \end{aligned} \tag{50}$$

for constants $c_{\delta_2}, c_J > 0$, the first inequality is due to Lemma 7 and $e_t = \|\mathcal{A}^* \mathcal{A}(U_t V_t^\top - \Sigma)\|_F$.

Therefore we can guarantee that

$$\begin{aligned} \|J_{t+1}\|_F &\leq (1 - \eta + c_{\delta_1}\eta)\|J_t\|_F + \eta c_{\delta_2} t e_t \\ &\leq (1 - \beta)^{t+1}\|J_0\|_F + c_{\delta_2} \eta \underbrace{\sum_{i=0}^t (1 - \beta)^i (t - i) e_{t-i}}_{\varpi_t^J}. \end{aligned} \tag{51}$$

where $c_{\delta_2}\varpi_t^J$ can be seen as a small perturbation. Similarly, we have

$$\|K_{t+1}\|_F \le (1 - \eta + c_{\delta_1}\eta)\|K_t\|_F + \eta c_{\delta_2} t e_t$$

$$\le (1 - \beta)^{t+1}\|K_0\|_F + c_{\delta_2}\,\eta \underbrace{\sum_{i=0}^{t}(1 - \beta)^i(t - i)e_{t-i}}_{\varpi_t^K}. \tag{52}$$

On the other hand we have

$$U_{t+1}V_t^\top = (1 - \eta)\begin{bmatrix}\hat{U}_t \\ J_t\end{bmatrix}[\hat{V}_t^\top, K_t^\top] + \eta\begin{bmatrix}\Sigma_r\hat{V}_t\left(\hat{V}_t^\top\hat{V}_t + K_t^\top K_t\right)^\dagger \\ 0\end{bmatrix}[\hat{V}_t^\top, K_t^\top]$$

$$+ \eta\begin{bmatrix}E_{U_t} \\ E_{J_t}\end{bmatrix}[\hat{V}_t^\top, K_t^\top], \tag{53}$$

which indicates that

$$\hat{U}_{t+1}\hat{V}_{t+1}^\top - \Sigma_r = (1 - \eta)^2(\hat{U}_t\hat{V}_t^\top - \Sigma_r) + \eta\left[\hat{U}_{t+1}\left(\hat{U}_{t+1}^\top\hat{U}_{t+1} + J_{t+1}^\top J_{t+1}\right)^\dagger\hat{U}_{t+1}^\top - I\right]\Sigma_r$$

$$+ \eta\hat{U}_{t+1}E_{V_t}^\top + (1 - \eta)\eta\Sigma_r\left[\hat{V}_t\left(\hat{V}_t^\top\hat{V}_t + K_t^\top K_t\right)^\dagger\hat{V}_t^\top - I\right] + (1 - \eta)\eta E_{U_t}\hat{V}_t^\top. \tag{54}$$

## A.2 Proofs of the lemmas

### A.2.1 Proof of Lemma 1

*Proof.* According to Eq. (9), we know that

$$\psi(X_{t+\frac{1}{2}}) = \frac{1}{2}\|\mathcal{B}(X_{t+\frac{1}{2}}, X_{t+\frac{1}{2}}) - \boldsymbol{b}\|_2^2$$

$$= \frac{1}{2}\|\mathcal{B}(X_t, X_t) - \eta\mathcal{B}(X_t, \hat{\Delta}(X_t)) - \eta\mathcal{B}(\hat{\Delta}(X_t), X_t) + \eta^2\mathcal{B}(\hat{\Delta}(X_t), \hat{\Delta}(X_t)) - \boldsymbol{b}\|_2^2$$

$$= \frac{1}{2}\|\eta\mathcal{B}(\hat{\Delta}(X_t), X_t) - (\mathcal{B}(X_t, X_t) - \boldsymbol{b})\|_2^2 \tag{55}$$

$$= \frac{1}{2}\|\mathcal{B}(X_t, X_t) - \boldsymbol{b}\|_2^2 + \frac{\eta^2}{2}\|\mathcal{B}(\hat{\Delta}(X_t), X_t)\|_2^2 - \eta\langle\mathcal{B}(\hat{\Delta}(X_t), X_t), \mathcal{B}(X_t, X_t) - \boldsymbol{b}\rangle.$$

Note that

$$\langle\mathcal{B}(\hat{\Delta}(X_t), X_t), \mathcal{B}(X_t, X_t) - \boldsymbol{b}\rangle = \|\hat{\mathcal{B}}(\Delta(X_t), X_t)\|_F^2, \tag{56}$$

and

$$\|\mathcal{B}(\hat{\Delta}(X_t), X_t)\|_2^2 \le \varrho\|\hat{\mathcal{B}}(\hat{\Delta}(X_t), X_t)\|_F^2, \tag{57}$$

where $\varrho = \|\mathcal{A}\|_2^2 \le (1 + \delta)$ which is due to the RIP condition. Thus we have

$$\psi(X_{t+\frac{1}{2}}) \le \psi(X_t) - \frac{2\eta - (1 + \delta)\eta^2}{2}\|\hat{\mathcal{B}}(\hat{\Delta}(X_t), X_t)\|_F^2. \tag{58}$$

Similarly, we can prove that

$$\psi(X_{t+1}) \le \psi(X_{t+\frac{1}{2}}) - \frac{2\eta - (1 + \delta)\eta^2}{2}\|\hat{\mathcal{B}}(X_{t+\frac{1}{2}}, \hat{\Delta}(X_{t+\frac{1}{2}}))\|_F^2. \tag{59}$$

Therefore we finish our proof. $\square$

### A.2.2 Proof of the Lemma 2.

*Proof.* Note that the optimal solution in Eq. (13) satisfies

$$\begin{bmatrix}I \\ \boldsymbol{0}\end{bmatrix}P\Delta X_t^\top QQ^\top X_t - \begin{bmatrix}I \\ \boldsymbol{0}\end{bmatrix}\mathcal{A}^*(\mathcal{B}(X_t, X_t) - \boldsymbol{b})Q^\top X_t = 0, \Delta \in \mathcal{C}. \tag{60}$$

While the solution of Eq. (9) is

$$\hat{\Delta}(X_t) = \begin{bmatrix} I \\ \mathbf{0} \end{bmatrix} \mathcal{A}^*(\mathcal{B}(X_t, X_t) - \boldsymbol{b})Q^\top X_t (X_t^\top Q Q^\top X_t^\top)^\dagger, \tag{61}$$

which satisfies

$$\begin{bmatrix} I \\ \mathbf{0} \end{bmatrix} P\Delta X_t^\top Q Q^\top X_t - \begin{bmatrix} I \\ \mathbf{0} \end{bmatrix} \mathcal{A}^*(\mathcal{B}(X_t, X_t) - \boldsymbol{b})Q^\top X_t = 0. \tag{62}$$

It is easy to verify that the matrix $\tilde{\Delta}(X_t) = \begin{bmatrix} P\hat{\Delta}(X_t) \\ P\hat{\Delta}(X_t) \end{bmatrix}$ satisfies Eq. (60), thus it is the solution of the problem (13). $\qquad\square$

### A.2.3 Proof of the Lemma 3.

Before proving the Lemma 3, we first define the angle between the column space of $VU^\top - \Sigma^\top$ and that of $V$ as

$$\cos\theta_v := \frac{\|(UV^\top - \Sigma)VV^\dagger\|_F}{\|UV^\top - \Sigma\|_F}, \tag{63}$$

and similarly

$$\cos\theta_u := \frac{\|(VU^\top - \Sigma)UU^\dagger\|_F}{\|UV^\top - \Sigma\|_F}, \tag{64}$$

where $\dagger$ stands for the pseudo-inverse. We note that the $\cos\theta_u$ in Eq. (63) and Eq. (64) are well defined and are equivalent to the following values.

**Proposition 1.** *For any $U, V \in \mathbb{R}^{n\times d}$ and $\Sigma \in \mathbb{R}^{n\times n}$, the $\cos\theta_v$ and $\cos\theta_u$ in Eq. (63) and Eq. (64) have the following equivalent formulations*

$$\cos\theta_v = \max_{\|Y\|_F=1} \frac{\langle UV^\top - \Sigma, YVV^\dagger \rangle}{\|UV^\top - \Sigma\|_F \|YVV^\dagger\|_F}, \tag{65}$$

*and*

$$\cos\theta_u = \max_{\|Y\|_F=1} \frac{\langle VU^\top - \Sigma, YUU^\dagger \rangle}{\|UV^\top - \Sigma\|_F \|YUU^\dagger\|_F}, \tag{66}$$

*where $Y \in \mathbb{R}^{n\times n}$.*

Now we present the proof of Lemma 3.

*Proof.* Given the $\cos\theta_v$ and $\cos\theta_u$ , one can immediately writes

$$\|(UV^\top - \Sigma)VV^\dagger\|_F = \cos\theta_v \|UV^\top - \Sigma\|_F, \tag{67}$$

and

$$\|(VU^\top - \Sigma)UU^\dagger\|_F = \cos\theta_u \|UV^\top - \Sigma\|_F, \tag{68}$$

which indicate that the $\cos\theta_v$ and $\cos\theta_u$ provide estimate to the $\sqrt{\tau_1}$ and $\sqrt{\tau_2}$ in Lemma 5. Together with the Lemma 4, Assumption 1 and Definition 1, we can establish that

$$\begin{aligned} \|\hat{\mathcal{B}}(\hat{\Delta}(X_t), X_t)\|_F^2 &\geq (1 - \delta_c)\|(U_tV_t^\top - \Sigma)V_tV_t^\dagger\|_F^2 \\ &= (1 - \delta_c)\cos^2\theta_v\|U_tV_t^\top - \Sigma\|_F \\ &\geq \frac{1 - \delta_c}{1 + \delta}\cos^2\theta_v^t[\psi(X_t) - \psi^*]. \end{aligned} \tag{69}$$

Similarly, we have

$$\|\hat{\mathcal{B}}(X_{t+\frac{1}{2}}, \hat{\Delta}(X_{t+\frac{1}{2}}))\|_F^2 \geq \frac{1 - \delta_c}{1 + \delta}\cos^2\theta_u^{t+1}[\psi(X_{t+\frac{1}{2}}) - \psi^*]. \tag{70}$$

According to Lemma 5, we know that $\cos^2\theta_v^t \geq \tau_1$ and $\cos^2\theta_u^t \geq \tau_2$, $\forall t > 0$ and $\max\{\tau_1, \tau_2\} > 0$. Thus we finish the proof. $\qquad\square$

### A.2.4 Proof of the Lemma 4.

We first prove the following lemma and the Lemma 4 is a natural result of the following lemma.

**Lemma 6.** *If the sensing operator $\mathcal{A}(\cdot)$ satisfies the RIP condition with constant $\delta$, then we have the following result*

$$\|\mathcal{A}^*\mathcal{A}(X)Y\|_F \geq (1-\delta_c)\|XY\|_F \tag{71}$$

*where $X, Y \in \mathbb{R}^{n \times n}$ and $0 < \delta_c < 1$.*

*Proof.* We have the following inequality

$$
\begin{aligned}
\|\mathcal{A}^*\mathcal{A}(X)Y\|_F &= \|\sum_{i=1}^{m}\langle A_i, X\rangle A_i Y\|_F \\
&= \max_{\|Z\|_F=1} \left\langle \sum_{i=1}^{m}\langle A_i, X\rangle A_i Y, Z \right\rangle \\
&= \max_{\|Z\|_F=1} \left\langle \sum_{i=1}^{m}\langle A_i, X\rangle A_i, ZY^\top \right\rangle \\
&= \max_{\|Z\|_F=1} \sum_{i=1}^{m}\langle A_i, X\rangle\langle A_i, ZY^\top\rangle \\
&\overset{\textcircled{1}}{\geq} \frac{1}{\sqrt{n}}\sum_{i=1}^{m}\langle A_i, X\rangle\langle A_i, Y^\top\rangle \\
&\geq \frac{1}{\sqrt{n}}\sum_{i=1}^{m}\sigma_{\min}^2(A_i)\|X\|_F\|Y\|_F
\end{aligned}
\tag{72}
$$

where the inequality $\textcircled{1}$ is obtained by setting $Z = \frac{1}{\sqrt{n}}I$ and the last inequality is due to $\langle A, B\rangle \geq \sigma_{\min}(A)\|B\|_F$. Then we obtain

$$\|\mathcal{A}^*\mathcal{A}(X)Y\|_F \geq \frac{1}{\sqrt{n}}\sum_{i=1}^{m}\sigma_{\min}^2(A_i)\|XY\|_F \tag{73}$$

With the help of the Theorem 2.7.5 in [44], we know that $\exists c > 0$ with high probability $\sigma_{\min}(A_i) \geq c/\sqrt{n}$. Thus we have

$$\frac{1}{\sqrt{n}}\sum_{i=1}^{m}\sigma_{\min}^2(A_i) \geq mc^2/n^{3/2}, \tag{74}$$

According to Lemma 1 in [21], we know that if $m \geq Cnr/\delta^2$ for sufficiently large universal constant $C$, then $\mathcal{A}(\cdot)$ satisfies the RIP condition with constant $\delta$. Thus we obtain

$$\frac{1}{\sqrt{n}}\sum_{i=1}^{m}\sigma_{\min}^2(A_i) \geq mc^2/n^{3/2} \geq \frac{Cc^2r}{\delta^2\sqrt{n}} \geq (1-\delta_c) \tag{75}$$

for some $0 < \delta_c < 1$. We complete the proof by combining Eq. (73) and Eq. (75). $\qquad\square$

### A.2.5 Proof of the Lemma 5

*Proof.* Let $E_t = U_t V_t^\top - \Sigma$, we have

$$
\begin{aligned}
\cos\theta_v^t &= \frac{\|(U_t V_t^\top - \Sigma) V_t V_t^\dagger\|_F}{\|E_t\|_F} = \frac{\|E_t V_t (V_t^\top V_t)^{-\frac{1}{2}}\|_F}{\|E_t\|_F} \\[2mm]
&= \max_{\|Y\|_F=1} \frac{\left\langle E_t V_t (V_t^\top V_t)^{-\frac{1}{2}}, Y \right\rangle}{\|E_t\|_F} \\[2mm]
&= \max_{\|Z(V_t^\top V_t)^{\frac{1}{2}}\|_F=1} \frac{\left\langle E_t, Z V_t^\top \right\rangle}{\|E_t\|_F} = \max_{\|Z V_t^\top\|_F=1} \frac{\left\langle E_t, Z V_t^\top \right\rangle}{\|E_t\|_F} \\[2mm]
&= \max_{\|Z V_t^\top\|_F=1} \frac{\left\langle \begin{bmatrix} \hat{U}_t \hat{V}_t^\top - \Sigma_r & \hat{U}_t K^\top \\ J_t \hat{V}_t^\top & J_t K_t^\top \end{bmatrix}, \begin{bmatrix} Z_1 \hat{V}_t^\top & Z_1 K_t^\top \\ Z_2 \hat{V}_t^\top & Z_2 K_t^\top \end{bmatrix} \right\rangle}{\|E_t\|_F} \\[2mm]
&= \max_{\|Z V_t^\top\|_F=1} \frac{\left\langle \begin{bmatrix} \hat{U}_t \hat{V}_t^\top - \Sigma_r \\ J_t \hat{V}_t^\top \end{bmatrix}, \begin{bmatrix} Z_1 \hat{V}_t^\top \\ Z_2 \hat{V}_t^\top \end{bmatrix} \right\rangle + \left\langle \begin{bmatrix} \hat{U}_t K_t^\top \\ J_t K_t^\top \end{bmatrix}, \begin{bmatrix} Z_1 K_t^\top \\ Z_2 K_t^\top \end{bmatrix} \right\rangle}{\|E_t\|_F} \\[2mm]
&= \max_{\|Z V_t^\top\|_F=1} \frac{\left\langle [\hat{U}_t \hat{V}_t^\top - \Sigma_r], Z_1 \hat{V}_t^\top \right\rangle}{\|E_t\|_F} + \frac{\left\langle J_t V_t^\top, Z_2 V_t^\top \right\rangle}{\|E_t\|_F} + \frac{\left\langle \hat{U}_t K_t^\top, Z_1 K_t^\top \right\rangle}{\|E_t\|_F} \\[2mm]
&\geq \max_{\|Z V_t^\top\|_F=1} \frac{\left\langle [\hat{U}_t \hat{V}_t^\top - \Sigma_r], Z_1 \hat{V}_t^\top \right\rangle + \left\langle J_t V_t^\top, Z_2 V_t^\top \right\rangle}{\|E_t\|_F} - \left| \frac{\left\langle \hat{U}_t K_t^\top, Z_1 K_t^\top \right\rangle}{\|E_t\|_F} \right| \\[2mm]
&\geq \max_{\|Z V_t^\top\|_F=1} \frac{\left\langle [\hat{U}_t \hat{V}_t^\top - \Sigma_r], Z_1 \hat{V}_t^\top \right\rangle + \left\langle J_t V_t^\top, Z_2 V_t^\top \right\rangle}{\|E_t\|_F} - \|Z_1 K_t^\top\|_F \frac{\|\hat{U}_t K_t^\top\|_F}{\|E_t\|_F} \\[2mm]
&\geq \frac{\|J_t V_t^\top\|_F}{\|E_t\|_F} \geq \frac{\|V_t\|_F \sigma_{\min}(J_t)}{\|E_t\|_F}.
\end{aligned}
\tag{76}
$$

Similarly, we have

$$
\begin{aligned}
\cos\theta_u^t &\geq \max_{\|Z U_t^\top\|_F=1} \frac{\left\langle [\hat{V}_t \hat{U}_t^\top - \Sigma_r], Z_1 \hat{U}_t^\top \right\rangle + \left\langle K_t U_t^\top, Z_2 U_t^\top \right\rangle}{\|E_t\|_F} - \|Z_1 J_t^\top\|_F \frac{\|J_t \hat{V}_t^\top\|_F}{\|E_t\|_F} \\[2mm]
&\geq \frac{\|U_t\|_F \sigma_{\min}(K_t)}{\|E_t\|_F}.
\end{aligned}
\tag{77}
$$

To estimate the lower-bound of $\cos\theta_u^t$ and $\cos\theta_v^t$, according to Eq. (76) and Eq. (77), we can divide the whole iterations into three stages based on the value of $\sigma_{\min}(K_t)$ and $\sigma_{\min}(J_t)$.

**Stage I:** In the first stage, there exists an iteration count $T_1$ such that $\sigma_{\min}(K_{T_1}) \geq \zeta_1$ and $\sigma_{\min}(J_T) \geq \zeta_1$, for small $\zeta_1 > 0$. Since $\mathcal{A}(\cdot)$ satisfies the RIP condition with sufficiently small $\delta$, we know that the $E_{K_t}, E_{J_t}$ in Eq. (44) are akin to small perturbations. Therefore the minimum singular value of $J_t$ and $K_t$ are decreasing under the Assumption 2, and one can guarantee that $\sigma_{\min}(K_t) \geq \zeta_1$ and $\sigma_{\min}(J_t) \geq \zeta_1$, $\forall t < T_1$. Then the value $\zeta_1$ provides lower-bound for the values of $\cos\theta_u^t \geq \tau_1 = \frac{d\zeta_1^2}{\|E_0\|_F}$ and $\cos\theta_v^t \geq \tau_2 = \frac{d\zeta_1^2}{\|E_0\|_F}$, for $t < T_1$. Meanwhile, according to the update in Eq. (44) and the sufficiently small RIP constant $\delta$, $E_{J_t}$ and $E_{K_t}$ can therefore be small enough such that $\sigma_{\min}(K_t)$ and $\sigma_{\min}(J_t)$ decrease almost linearly in stage I with $T_1 = O(\log\frac{1}{\zeta_1})$ for very small $\zeta_1$.

**Stage II:** In the second stage, if $\sigma_{\min}(K_t) < \zeta_1$ and $\sigma_{\min}(J_t) < \zeta_1$, for $T_2 > t > T_1$ for some $T_2$, we can use the following inequalities to lower-bound the $\cos\theta_u^t$ and $\cos\theta_v^t$.

$$
\cos\theta_u^t \geq \max_{\|Z U_t^\top\|_F=1} \frac{\left\langle [\hat{V}_t \hat{U}_t^\top - \Sigma_r], Z_1 \hat{U}_t^\top \right\rangle + \left\langle K_t U_t^\top, Z_2 U_t^\top \right\rangle}{\|E_t\|_F} - \|Z_1 J_t^\top\|_F \frac{\|J_t \hat{V}_t^\top\|_F}{\|E_t\|_F},
\tag{78}
$$

$$\cos\theta_v^t \geq \max_{\|ZV_t^\top\|_F=1} \frac{\left\langle \left[\hat{U}_t\hat{V}_t^\top - \Sigma_r\right], Z_1\hat{V}_t^\top\right\rangle + \left\langle J_tV_t^\top, Z_2V_t^\top\right\rangle}{\|E_t\|_F} - \|Z_1K_t^\top\|_F\frac{\|\hat{U}_tK_t^\top\|_F}{\|E_t\|_F}, \quad (79)$$

which come from Eq. (77) and Eq. (76).

By Lemma 7, we know that $\sigma_{\min}(\hat{U}_t) > 0$ and $\sigma_{\min}(\hat{V}_t) > 0$, for $t > T$. Together with Eq. (78) and Eq. (79) we have

$$\cos\theta_u^t \geq \frac{\|\hat{V}_t\hat{U}_t^\top - \Sigma_r\|_F\|Z_1^u\hat{U}_t^\top\|_F + \|K_tU_t^\top\|_F\|Z_2^uU_t^\top\|_F}{\|E_t\|_F} - \|Z_1^uJ_t^\top\|_F\frac{\|J_t\hat{V}_t^\top\|_F}{\|E_t\|_F}, \quad (80)$$

where $\|Z_1^u\hat{U}_t^\top\|_F^2 + \|Z_1^uJ_t\|_F^2 + \|Z_2^uU_t^\top\|_F^2 = 1$.

$$\cos\theta_v^t \geq \frac{\|\hat{U}_t\hat{V}_t^\top - \Sigma_r\|_F\|Z_1^v\hat{V}_t^\top\|_F + \|J_tV_t^\top\|_F\|Z_2^vV_t^\top\|_F}{\|E_t\|_F} - \|Z_1^vK_t^\top\|_F\frac{\|\hat{U}_tK_t^\top\|_F}{\|E\|_F}, \quad (81)$$

where $\|Z_1^v\hat{V}_t^\top\|_F^2 + \|Z_2^vV^\top\|_F^2 + \|Z_1^vK_t^\top\|_F^2 = 1$. The Eq. (80) and Eq. (81) further imply that

$$\cos\theta_u^t \geq \sqrt{1 - \|Z^uJ_t\|_F^2}\frac{\left\|\begin{bmatrix}\hat{V}_t\hat{U}_t^\top - \Sigma_r \\ K_t\hat{U}_t^\top\end{bmatrix}\right\|_F}{\|E_t\|_F} - \|Z^uJ_t\|_F, \quad (82)$$

and

$$\cos\theta_v^t \geq \sqrt{1 - \|Z^vK_t\|_F^2}\frac{\left\|\begin{bmatrix}\hat{U}_t\hat{V}_t^\top - \Sigma_r \\ J_t\hat{V}_t^\top\end{bmatrix}\right\|_F}{\|E_t\|_F} - \|Z^vK_t\|_F. \quad (83)$$

Meanwhile

$$\begin{aligned}\|E_t\|_F^2 &= \|J_t\hat{V}_t^\top\|_F^2 + \|\hat{U}_tK_t^\top\|_F^2 + \|\hat{U}_t\hat{V}_t^\top - \Sigma_r\|_F^2 + \|J_tK_t^\top\|_F^2 \\ &\leq 2\left(\max\{\|J_t\hat{V}_t^\top\|_F^2, \|\hat{U}_tK_t^\top\|_F^2\} + \max\{\|\hat{U}_t\hat{V}_t^\top - \Sigma_r\|_F^2, \|J_tK_t^\top\|_F^2\}\right).\end{aligned} \quad (84)$$

According to Eq. (51), Eq. (52) and Eq. (54), we know that $\|\hat{U}_t\hat{V}_t^\top - \Sigma_r\|_F^2 \geq \|J_tK_t^\top\|_F^2$, if $\|\hat{U}_0\hat{V}_0^\top - \Sigma_r\|_F^2 \geq \|J_0K_0^\top\|_F^2$ which is true for random Gauss initialization with small value $c_0$ in Assumption 2. Thus, we have $\|E_t\|_F^2 \leq 2\left(\max\{\|J_t\hat{V}_t^\top\|_F^2, \|\hat{U}_tK_t^\top\|_F^2\} + \|\hat{U}_t\hat{V}_t^\top - \Sigma_r\|_F^2\right)$ and

$$\max\left\{\frac{\left\|\begin{bmatrix}\hat{U}_t\hat{V}_t^\top - \Sigma_r \\ J_t\hat{V}_t^\top\end{bmatrix}\right\|_F}{\|E_t\|_F}, \frac{\left\|\begin{bmatrix}\hat{V}_t\hat{U}_t^\top - \Sigma_r \\ K_t\hat{U}_t^\top\end{bmatrix}\right\|_F}{\|E_t\|_F}\right\} \geq \sqrt{2}/2. \quad (85)$$

Without loss of generality, we assume $\|J_t\hat{V}_t^\top\|_F \geq \|K_t\hat{U}_t^\top\|_F$, then we consider Eq. (83), such that

$$\cos\theta_v^t \geq \sqrt{1 - \|Z^vK_t\|_F^2}\frac{\sqrt{2}}{2} - \|Z^vK_t\|_F. \quad (86)$$

If $\|Z^vK_t\|_F \leq 1/3$, then we can guarantee that $\cos\theta_v^t \geq 1/3$. Meanwhile, as $\|Z^vK_t\|_F$ decreases, the lower bound of $\cos\theta_v^t$ increases. Now we prove that in a period $[T_1, T_2]$, $\|Z^vK_t\|_F \leq 1/3$ and in this period, the upper bound of $\|Z^vK_t\|_F$ decreases monotonically.

Note that $\sigma_{\min}^2(\hat{V}_t)\|Z^v\|_F^2 \leq \|Z^v\hat{V}_t^\top\|_F^2 \leq 1$, then $\|Z^v\|_F^2 \leq 1/\sigma_{\min}^2(\hat{V})$. Moreover, according to Eq. (51) we know

$$\|K_{t+1}\|_F \leq (1-\beta)^t\|K_0\|_F + c_{\delta_2}\varpi_t^K, \quad (87)$$

for $\beta < 1$ and $c_{\delta_2}$ is related to the RIP condition $\delta$ and can be sufficiently small. In consequence,

$$\|Z^vK_t^\top\|_F \leq (1-\beta)^t\|K_0\|_F/\sigma_{\min}(\hat{V}_t) + c_{\delta_2}\varpi_t^K/\sigma_{\min}(\hat{V}_t). \quad (88)$$

According to Lemma 7, one can guarantee that

$$\|Z^vK_t^\top\|_F \leq t(1-\beta)^t\|K_0\|_F\tilde{c}_v + t\varpi_t^K\tilde{c}_v. \quad (89)$$

Note that

$$\varpi_t^K \le \eta \sum_{i=0}^{t} (1-\beta)^i (t-i) e_{t-i} \le c_0 \eta (1-c_\gamma)^{T_1}, \tag{90}$$

for constant $c_0$ and $c_\gamma < 1$ as long as $t > T_1$. The above inequality is due to that in stage I, the objective function decreases linearly such that $e_t \le c_\tau (1-\hat{c}_\gamma)^t$ for $t \le T_1$. Meanwhile, we can select $\zeta_1$ in stage I to be small with $T_1 = O(\log \frac{1}{\zeta_1})$ such that

$$\|Z^v K_t^\top\|_F \le t(1-\beta)^t \|K_0\|_F \tilde{c}_v + t c_0 \eta (1-c_\gamma)^{T_1} \le 1/3, \tag{91}$$

for $T_2 > t > T_1$ for some $T_2$.

**Stage III:** Note that as long as $\|Z^v K_t^\top\|_F \le 1/3$, we can prove that the $\max\{\cos\theta_v^t, \cos\theta_u^t\}$ in stage II is lower-bounded by a constant $\tau_s \ge 1/3$, which guarantees that $e_t$ decreases linearly in $t \in [0, T_2]$. In consequence $\varpi_t^K \le c_0 \eta (1-c_\gamma)^t$ and

$$\|Z^v K_t^\top\|_F \le \underbrace{t(1-\beta)^t \|K_0\|_F \tilde{c}_v + t\tilde{c}_v (1-c_\gamma)^t}_{\varkappa_t}. \tag{92}$$

Now we prove that $\varkappa_t \le 1/3$ for all $t > T_2$ such that $\max\{\cos\theta_v^t, \cos\theta_u^t\} \ge 1/3$, by setting proper $T_2$. It is easy to see that $\varkappa_t$ is a decreasing function when $t > \tilde{T}$ for certain $\tilde{T}$. As noted in stage II, we can set $T_2 > \tilde{T}$ such that

$$\|Z^v K_{T_2}^\top\|_F \le T_2 (1-\beta)^{T_2} \|K_0\|_F \tilde{c}_v + T_2 c_0 \eta (1-c_\gamma)^{T_1} \le 1/3, \tag{93}$$

which is achievable by setting $T_1$ to be large enough.

Together with all the three stages, we conclude that by setting $\tau_t^1, \tau_t^2$ in Eq. (26) as $\cos\theta_u^t$ and $\cos\theta_v^t$ respectively, we have that the results in Eq. (26) hold for $\max\{\tau_t^1, \tau_t^2\} \ge \tau > 0$. $\qquad\square$

### A.2.6   Proof of the Lemma 7

**Lemma 7.** *Suppose that the Assumption 2 and Assumption 1 hold. Let $\hat{U}_t$ and $\hat{V}_t$ be updated according to Eq. (42) and (43). Then, it follows that there exist constants $\tilde{c}_u$ and $\tilde{c}_v$ such that*

$$\sigma_r(\hat{U}_t) \ge \tilde{c}_u/t, \quad \sigma_r(\hat{V}_t) \ge \tilde{c}_v/t. \tag{94}$$

*Proof.* Note that

$$\begin{aligned}
\hat{U}_{t+1} &= (1-\eta)\hat{U}_t + \eta \Sigma_r \hat{V}_t \left(\hat{V}_t^\top \hat{V}_t + K_t^\top K_t\right)^\dagger + \eta E_{U_t} \\
&= (1-\eta)\hat{U}_t + \eta \Sigma_r \left(\hat{U}_t \hat{V}_t^\top\right)^\dagger \hat{U}_t \hat{V}_t^\top \hat{V}_t \left(\hat{V}_t^\top \hat{V}_t + K_t^\top K_t\right)^\dagger + \eta E_{U_t},
\end{aligned} \tag{95}$$

and

$$\begin{aligned}
\hat{V}_{t+1} &= (1-\eta)\hat{V}_t + \eta \Sigma_r \hat{U}_{t+1} \left(\hat{U}_{t+1}^\top \hat{U}_{t+1} + J_{t+1}^\top J_{t+1}\right)^\dagger + \eta E_{V_t} \\
&= (1-\eta)\hat{V}_t + \eta \Sigma_r (\hat{V}_t \hat{U}_{t+1}^\top)^\dagger \hat{V}_t \hat{U}_{t+1}^\top \hat{U}_{t+1} \left(\hat{U}_{t+1}^\top \hat{U}_{t+1} + J_{t+1}^\top J_{t+1}\right)^\dagger + \eta E_{V_t}.
\end{aligned} \tag{96}$$

Therefore we have the upper-bound of the operator norm of $\hat{U}_{t+1}$ as

$$\begin{aligned}
\|\hat{U}_{t+1}\|_2 &\le (1-\eta)\|\hat{U}_t\|_2 + \eta \underbrace{\left\|\Sigma_r \left(\hat{U}_t \hat{V}_t^\top\right)^\dagger\right\|_2}_{\tau_t} \|\hat{U}_t\|_2 \underbrace{\left\|\hat{V}_t^\top \hat{V}_t \left(\hat{V}_t^\top \hat{V}_t + K_t^\top K_t\right)^\dagger\right\|_2}_{\nu_t} + \eta\|E_{U_t}\|_2 \\
&\le (1 + \eta(\tau_t \nu_t - 1)) \|\hat{U}_t\|_2 + \eta\|E_{U_t}\|_2 \le (1 + \eta(\tau_t - 1)) \|\hat{U}_t\|_2 + \eta\|E_{U_t}\|_2 \\
&\le (1 + \eta|\tau_t - 1|(1+\delta_2)) \|\hat{U}_t\|_2 + \eta\|E_{U_t}\|_2,
\end{aligned} \tag{97}$$

where $\delta_2$ is due to the RIP condition in Assumption 1 and $\delta_2 \ll 1$ is sufficiently small. Meanwhile

$$
\begin{aligned}
\|\Sigma_r \left(\hat{U}_t \hat{V}_t^\top\right)^\dagger - I\|_2 &= \| \left(\Sigma_r - \hat{U}_t \hat{V}_t^\top\right) \left(\hat{U}_t \hat{V}_t^\top\right)^\dagger \|_2 \\
&\leq \|\Sigma_r - \hat{U}_t \hat{V}_t^\top\|_2 / \sigma_{\min}(\hat{U}_t \hat{V}_t^\top) \\
&\leq \|\Sigma_r - \hat{U}_t \hat{V}_t^\top\|_2 / c_\xi \\
&\leq \frac{(1 - \zeta_t \eta)^t}{c_\xi} \|\Sigma_r - \hat{U}_0 \hat{V}_0^\top\|_2,
\end{aligned}
\tag{98}
$$

where $c_\xi$ is due to Lemma (8). The last inequality is due to that as long as $\min\{\sigma_{\min}(K_t), \sigma_{\min}(J_t)\} > \zeta_t^c > 0$, one can always guarantee that $\|\Sigma_r - \hat{U}_t \hat{V}_t^\top\|_2 \leq (1 - \zeta_t \eta)^t \|\Sigma_r - \hat{U}_0 \hat{V}_0^\top\|_2$ for $\zeta_t > 0$. Therefore we have

$$
|\tau_t - 1| = O(c_\gamma^{-t}),
\tag{99}
$$

with $c_\gamma \geq 1$. Together with Eq. (97) we obtain

$$
\begin{aligned}
\|\hat{U}_{t+1}\|_2 &\leq (1 + \eta c_\zeta c_\gamma^{-t})\|\hat{U}_t\|_2 + \eta\|E_{U_t}\|_2 \\
&\leq \eta \sum_{i=0}^{t} \prod_{j=0}^{i} (1 + \eta c_\zeta c_\gamma^{-i})\|E_{U_{t-i}}\|_2 + \prod_{j=0}^{i}(1 + \eta c_\zeta c_\gamma^{-i})\|U_0\|_2 \\
&\stackrel{\text{(a)}}{\leq} \eta \sum_{i=0}^{t} \prod_{j=0}^{i} (1 + \eta c_\zeta c_\gamma^{-i})\|E_{U_0}\|_2 + \prod_{j=0}^{i}(1 + \eta c_\zeta c_\gamma^{-i})\|U_0\|_2 \\
&\stackrel{\text{(b)}}{\leq} C_E t + C_U \leq c_u t
\end{aligned}
\tag{100}
$$

for constant $C_E, C_U, c_u$. The inequality ⓐ is due to the monotonically decreasing of $\|E_{U_t}\|_2$, which is obtained from Eq. (47) and Eq. (48). The inequality ⓑ is due to $\prod_{j=0}^{t}(1 + \eta c_\zeta c_\gamma^{-i}) \leq C_p$ for constant $C_p$. Similarly, one can guarantee that

$$
\|\hat{V}_t\|_2 \leq c_v t
\tag{101}
$$

for constant $c_v$. In consequence, we obtain

$$
\sigma_r(\hat{U}_t) \leq \|\hat{U}_t\|_2 \leq c_u t, \ \ \sigma_r(\hat{V}_t) \leq \|\hat{V}_t\|_2 \leq c_v t.
\tag{102}
$$

According to Lemma 8, we know that the $\sigma_r(\hat{U}_t \hat{V}_t^\top) \geq c_\xi$, together with Eq. (102) we can guarantee that

$$
\sigma_r(\hat{U}_t) \geq \frac{c_\xi}{t c_u}, \ \ \sigma_r(\hat{V}_t) \geq \frac{c_\xi}{t c_v},
\tag{103}
$$

where we use the fact $\sigma_r(\hat{U} \hat{V}^\top) \leq \|\hat{U}\|_2 \sigma_r(\hat{V})$. Let $\tilde{c}_u = c_\xi / c_u$ and $\tilde{c}_v = c_\xi / c_v$, we thus finish the proof. $\qquad \square$

### A.2.7 Proof of the Lemma 8

**Lemma 8.** *Let $\hat{U}_t$ and $\hat{V}t$ be updated according to Eq. (42) and Eq. (43), respectively. Then, there exist constants $c_\xi > 0$ such that*

$$
\sigma_r(\hat{U}_t \hat{V}_t^\top) \geq c_\xi, \forall t > 0.
\tag{104}
$$

*Proof.* Note that

$$
\sigma_r(\underbrace{\hat{U}_{t+1}[\hat{U}_{t+1}^\top \hat{U}_{t+1} + J_{t+1}^\top J_{t+1}]^\dagger \hat{U}_{t+1}^\top}_{P_{t+1}} \Sigma_r) \geq \sigma_r(P_{t+1})\sigma_r(\Sigma_r),
\tag{105}
$$

and

$$
\sigma_r(\Sigma_r \hat{V}_{t+1}\underbrace{[\hat{V}_{t+1}^\top \hat{V}_{t+1} + K_{t+1}^\top K_{t+1}]^\dagger \hat{V}_{t+1}^\top}_{Q_{t+1}}) \geq \sigma_r(Q_{t+1})\sigma_r(\Sigma_r).
\tag{106}
$$

According to Eq. (42) and Eq. (43), we have

$$
\begin{aligned}
\hat{U}_{t+1}\hat{V}_{t+1}^\top &= (1-\eta)^2\hat{U}_t\hat{V}_t^\top + \eta\hat{U}_{t+1}\left(\hat{U}_{t+1}^\top\hat{U}_{t+1} + J_{t+1}^\top J_{t+1}\right)^\dagger \hat{U}_{t+1}^\top\Sigma_r \\
&\quad + \eta\hat{U}_{t+1}E_{V_t}^\top + (1-\eta)\eta\Sigma_r\hat{V}_t\left(\hat{V}_t^\top\hat{V}_t + K_t^\top K_t\right)^\dagger \hat{V}_t^\top + (1-\eta)\eta E_{U_t}\hat{V}_t^\top \\
&= (1-\eta)^{2t}\hat{U}_0\hat{V}_0^\top + \underbrace{\eta\sum_{i=0}^{t}(1-\eta)^{2i}P_{t+1-i}\Sigma_r + \eta(1-\eta)\sum_{i=0}^{t}(1-\eta)^{2i}\Sigma_r Q_{t-i}}_{\mathcal{H}} \quad (107) \\
&\quad + \underbrace{\eta\sum_{i=0}^{t}(1-\eta)^{2i}\hat{U}_{t+1-i}E_{V_{t-i}}^\top + \eta(1-\eta)\sum_{i=0}^{t}(1-\eta)^{2i}E_{U_{t-i}}\hat{V}_{t-i}^\top}_{\mathcal{Z}} .
\end{aligned}
$$

Meanwhile

$$
\sigma_r(\mathcal{H}) \geq \eta\sum_{i=0}^{t}(1-\eta)^{2i}\sigma_r(P_{t+1-i}\Sigma_r) + \eta(1-\eta)\sum_{i=0}^{t}(1-\eta)^{2i}\sigma_r(\Sigma_r Q_{t-i}). \qquad (108)
$$

The fact that $\sigma_r(P_{t+1}) \geq \sigma_r(P_0)$ and $\sigma_r(Q_{t+1}) \geq \sigma_r(Q_0)$ gives to

$$
\sigma_r(\mathcal{H}) \geq \eta\left(\sigma_r(P_0) + (1-\eta)\sigma_r(Q_0)\right)\sigma_r(\Sigma_r). \qquad (109)
$$

Moreover, according the RIP condition in Assumption 1, the following inequalities hold

$$
\|E_{U_t}\hat{V}_t^\top\|_2 \leq \delta_2\|\hat{U}_0\hat{V}_0^\top - \Sigma_r\hat{V}_0(\hat{V}_0^\top\hat{V}_0 + K_0^\top K_0)^\dagger\hat{V}_0^\top\|_2 = \delta_2 c_\nu^u, \qquad (110)
$$

and

$$
\|\hat{U}_{t+1}E_{V_t}^\top\|_2 \leq \delta_2\|\hat{U}_0\hat{V}_0^\top - \hat{U}_{t+1}[\hat{U}_{t+1}^\top\hat{U}_{t+1} + J_{t+1}^\top J_{t+1}]^\dagger\hat{U}_{t+1}^\top\Sigma_r\|_2 = \delta_2 c_\nu^v \qquad (111)
$$

for sufficiently small $\delta_2$, and $\|\mathcal{Z}\|_2 \leq \frac{1}{2-\eta}\delta_2 c_\nu^v + \frac{1-\eta}{2-\eta}\delta_2 c_\nu^u$.

Since $\varrho = (1-\eta)^{2t} \ll 1$ and $\delta_2$ are very small, now we can lower bound the $r$-th singular value of $\hat{U}_{t+1}\hat{V}_{t+1}^\top$ by

$$
\begin{aligned}
\sigma_r(\hat{U}_{t+1}\hat{V}_{t+1}^\top) &\geq \sigma_r(\mathcal{H}) - \varrho\sigma_1(\hat{U}_0\hat{V}_0^\top) - \|\mathcal{Z}\|_2 \\
&\geq \eta\left(\sigma_r(P_0) + (1-\eta)\sigma_r(Q_0)\right)\sigma_r(\Sigma_r) - \varrho\sigma_1(\hat{U}_0\hat{V}_0^\top) - \delta_2\left(\frac{1}{2-\eta}c_\nu^u + \frac{1-\eta}{2-\eta}c_\nu^v\right) \\
&\geq c_\xi,
\end{aligned}
$$
$$(112)$$

which is a simple result of the matrix perturbation theory and $c_\xi > 0$ is a universal constant. $\qquad\square$

### A.3 Proofs of the theorems

#### A.3.1 Proof of the Theroem 1

*Proof.* Since $M$ is rank one matrix, we have $U_s = \mathbf{0} \in \mathbb{R}^{n\times d}$ and $V_s = \mathbf{0} \in \mathbb{R}^{n\times d}$ correspondingly

$$
\hat{U} = U_s + \varepsilon N_u = \varepsilon N_u, \hat{V} = V_s + \varepsilon N_v = \varepsilon N_v. \qquad (113)
$$

Note that

$$
\begin{aligned}
\|\nabla g_{\hat{U}}\|_F^2 &= \|(\hat{U}\hat{V}^\top - M)\hat{U}\|_F^2 \\
&= \|(U_sV_s^\top - M + \underbrace{\varepsilon U_s N_v^\top + \varepsilon N_u V_s^\top + \varepsilon^2 N_u N_v^\top}_{\varepsilon Z})(U_s + \varepsilon N_u)\|_F^2 \\
&= \|(U_s U_s^\top - M + \varepsilon Z)\varepsilon N_u\|_F^2 \\
&= \|\varepsilon(U_s U_s^\top - M)N + \varepsilon^2 Z N_u\|_F^2 \\
&= o(\varepsilon)e_s + o(\varepsilon^2).
\end{aligned}
$$
$$(114)$$

Similarly we have $\|\nabla g_{\hat{V}}\|_F^2 = o(\varepsilon)e_s + o(\varepsilon^2)$ and therefore

$$\|\nabla g\|_F^2 = \|\nabla g_{\hat{U}}\|_F^2 + \|\nabla g_{\hat{V}}\|_F^2 = o(\varepsilon)e_s + o(\varepsilon^2).$$

Meanwhile

$$
\begin{aligned}
\|\nabla g_{\hat{U}}(\hat{V}^\top \hat{V})^{-\frac{1}{2}}\|_F^2 &= \|(\hat{U}\hat{V}^\top - M)\hat{V}\hat{V}^\dagger\|_F^2 \\
&= \|(U_s U_s^\top - M + \underbrace{\varepsilon U_s N^\top + \varepsilon N U_s^\top + \varepsilon^2 N N^\top}_{\varepsilon Z})N_v N_v^\dagger\|_F^2 \\
&= \|M N_v N_v^\dagger - \varepsilon^2 N N^\top N_v N_v^\dagger\|_F^2 \\
&= \|M N_v N_v^\dagger - \varepsilon^2 N N^\top\|_F^2.
\end{aligned}
\tag{115}
$$

Let $\mathcal{V}\hat{\Sigma}\mathcal{V}^\top = N_v N_v^\dagger$ be the SVD of $N_v N_v^\dagger$, then $\|M N_v N_v^\dagger\|_F^2 = \sigma_*^2\|\mathcal{V}^\top V^*\|_F^2 = e_s\|\mathcal{V}^\top V^*\|_F^2$ where $M = \sigma_* U^* V^{*\top}$ is the SVD of the rank-1 $M$ and $\mathcal{V}$ is the orthogonal basis of a random Gauss matrix $N_v$, then with high probability we have $\|\mathcal{V}^\top V^*\|_F^2 = \Theta(1)$. As a result, we obtain

$$\|\nabla g_{\hat{U}}(\hat{V}^\top \hat{V})^{-\frac{1}{2}}\|_F^2 = \Theta(1)e_s + o(\varepsilon^2).\tag{116}$$

Likewise, we have

$$\|\nabla g_{\hat{V}}(\hat{U}^\top \hat{U})^{-\frac{1}{2}}\|_F^2 = \Theta(1)e_s + o(\varepsilon^2).\tag{117}$$

As for the symmetric matrix sensing, we have

$$
\begin{aligned}
\|\nabla g_{\hat{U}}(\hat{U}^\top \hat{U} + \lambda I)^{-\frac{1}{2}}\|_F^2 &= \|(\hat{U}\hat{U}^\top - M)\hat{U}(\hat{U}^\top \hat{U} + \lambda I)^{-\frac{1}{2}}\|_F^2 \\
&= \|(\varepsilon^2 N N^\top - M)\varepsilon N(\varepsilon^2 N^\top N + \lambda I)^{-\frac{1}{2}}\|_F^2 \\
&= \|(\varepsilon^2 N N^\top - M)\varepsilon \Phi \Sigma_N (\varepsilon^2 \Sigma_N^2 + \lambda I)^{-\frac{1}{2}}\Psi^\top\|_F^2 \\
&= \|M\varepsilon \Phi \Sigma_N(\varepsilon^2 \Sigma_N^2 + \lambda I)^{-\frac{1}{2}}\Sigma_N \Psi^\top - \varepsilon^3 N N^\top N(\varepsilon^2 N^\top N + \lambda I)^{-\frac{1}{2}}\|_F^2,
\end{aligned}
\tag{118}
$$

where $N$ is random Gauss matrices that follows standard normal distribution and $N = \Phi \Sigma_N \Psi^\top$ is the SVD of $N$. Note that

$$
\begin{aligned}
\|M\varepsilon \Phi \Sigma_N(\varepsilon^2 \Sigma_N^2 + \lambda I)^{-\frac{1}{2}}\Psi^\top\|_F^2 &= \sigma_*^2 \|\varepsilon U^{*\top}\Phi \Sigma_N(\varepsilon^2 \Sigma_N^2 + \lambda I)^{-\frac{1}{2}}\|_F^2 \\
&= \Theta\left(\frac{\varepsilon^2}{\varepsilon^2 + \lambda/c}\right)e_s,
\end{aligned}
\tag{119}
$$

where $c > 0$ is a bounded constant which is related to the singular value of the matrix $N$, thus we have that

$$\|\nabla g_{\hat{U}}(\hat{U}^\top \hat{U} + \lambda I)^{-\frac{1}{2}}\|_F^2 = \Theta\left(\frac{\varepsilon^2}{\varepsilon^2 + \lambda/c}\right)e_s + o(\varepsilon^2).\tag{120}$$

Thus we finish our proof. □

### A.3.2   Proof of the Theorem 2

*Proof.* According to Lemma 1, we know that

$$
\begin{aligned}
\psi(X_{t+\frac{1}{2}}) &\leq \psi(X_t) - \ell\|\hat{\mathcal{B}}(\hat{\Delta}(X_t), X_t)\|_F^2, \\
\psi(X_{t+1}) &\leq \psi(X_{t+\frac{1}{2}}) - \ell\|\hat{\mathcal{B}}(X_{t+\frac{1}{2}}, \hat{\Delta}(X_{t+\frac{1}{2}}))\|_F^2,
\end{aligned}
\tag{121}
$$

where $\ell = (2\eta - (1+\delta)\eta^2)/2$. Together with Lemma 3 which shows

$$
\begin{aligned}
\|\hat{\mathcal{B}}(\hat{\Delta}(X_t), X_t)\|_F^2 &\geq \frac{1-\delta}{1+\delta}\tau_t^1[\psi(X_t) - \psi^*], \\
\|\hat{\mathcal{B}}(X_{t+\frac{1}{2}}, \hat{\Delta}(X_{t+\frac{1}{2}}))\|_F^2 &\geq \frac{1-\delta}{1+\delta}\tau_t^2[\psi(X_{t+\frac{1}{2}}) - \psi^*].
\end{aligned}
\tag{122}
$$

We arrived at

$$
\begin{aligned}
\psi(X_{t+\frac{1}{2}}) - \psi^* &\leq \psi(X_t) - \psi^* - \ell\|\hat{\mathcal{B}}(\hat{\Delta}(X_t), X_t)\|_F^2 \\
&\leq (1 - \ell\tau_t^1\frac{1-\delta}{1+\delta})[\psi(X_t) - \psi^*],
\end{aligned}
\tag{123}
$$

and

$$\psi(X_{t+1}) - \psi^* \leq \psi(X_{t+\frac{1}{2}}) - \psi^* - \ell\|\hat{\mathcal{B}}(X_{t+\frac{1}{2}}, \hat{\Delta}(X_{t+\frac{1}{2}}))\|_F^2$$
$$\leq (1 - \ell\tau_t^2 \frac{1-\delta}{1+\delta})[\psi(X_{t+\frac{1}{2}}) - \psi^*]. \tag{124}$$

Thus we hve

$$\psi(X_{t+1}) - \psi^* \leq (1 - \ell\tau_t^2 \frac{1-\delta}{1+\delta})(1 - \ell\tau_t^1 \frac{1-\delta}{1+\delta})[\psi(X_t) - \psi^*]$$
$$\leq (1 - \ell\tau \frac{1-\delta}{1+\delta})[\psi(X_t) - \psi^*], \tag{125}$$

where $\tau = \max\{\tau_t^1, \tau_t^2\} > 0$.

Meanwhile, if the RIP constant $\delta = 0$, which means the number of the sampling $m \to \infty$ and $\mathcal{A}(\cdot)$ becomes the identity operator, one can set $\eta = 1$. We know from the Eq. (32) and Eq. (33) that

$$\psi(X_{t+1}) - \psi^* = \frac{1}{2}\|U_{t+1}V_{t+1}^\top - M\|_F^2 = \frac{1}{2}\|M^\top U_{t+1}U_{t+1}^\dagger - M\|_F^2. \tag{126}$$

At the same time

$$U_{t+1}V_t^\top = MV_tV_t^\dagger. \tag{127}$$

Thus $M^\top U_{t+1}U_{t+1}^\dagger = M$, which indicates that

$$\psi(X_{t+1}) - \psi^* = \frac{1}{2}\|U_{t+1}V_{t+1}^\top - M\|_F^2,$$
$$= \frac{1}{2}\|M^\top U_{t+1}U_{t+1}^\dagger - M\|_F^2,$$
$$= 0[\frac{1}{2}\|M^\top U_tU_t^\dagger - M\|_F^2], \tag{128}$$
$$= 0[\psi(X_t) - \psi^*].$$

$\square$

